# Differential Privacy in Scalable General Kernel Learning via $K$-means Nyström Random Features

**Bonwoo Lee**        **Jeongyoun Ahn**[*]        **Cheolwoo Park**[*]

KAIST

Daejeon, 34141 South Korea

righthim@kaist.ac.kr, jyahn@kaist.ac.kr, parkcw2021@kaist.ac.kr

## Abstract

As the volume of data invested in statistical learning increases and concerns regarding privacy grow, the privacy leakage issue has drawn significant attention. Differential privacy has emerged as a widely accepted concept capable of mitigating privacy concerns, and numerous differentially private (DP) versions of machine learning algorithms have been developed. However, existing works on DP kernel learning algorithms have exhibited practical limitations, including scalability, restricted choice of kernels, or dependence on test data availability. We propose DP scalable kernel empirical risk minimization (ERM) algorithms and a DP kernel mean embedding (KME) release algorithm suitable for general kernels. Our approaches address the shortcomings of previous algorithms by employing Nyström methods, classical techniques in non-private scalable kernel learning. These methods provide data-dependent low-rank approximations of the kernel matrix for general kernels in a DP manner. We present excess empirical risk bounds and computational complexities for the scalable kernel DP ERM, KME algorithms, contrasting them with established methodologies. Furthermore, we develop a private data-generating algorithm capable of learning diverse kernel models. We conduct experiments to demonstrate the performance of our algorithms, comparing them with existing methods to highlight their superiority.

## 1 Introduction

As data collection and access continue to expand, protecting privacy has emerged as a crucial concern. *Differential privacy* (DP), introduced by Dwork et al. (2006b), stands as the current gold standard in data privacy. It provides a rigorous framework for privacy in statistical procedures, wherein noise is added proportional to the maximum deviation induced by the change of a single individual in the dataset. The field has seen significant advances in differentially private (DP) algorithms for machine learning. Notably, DP empirical risk minimization (ERM) has become a key area of research, attracting extensive attention (Chaudhuri et al., 2011; Kifer et al., 2012; Bassily et al., 2019; Wang et al., 2019; Feldman et al., 2020). These efforts focus on developing privacy-preserving statistical models with high generalization capabilities across various ERM problems, including regularized regression, logistic regression, support vector machines, and some other ERM problems equipped with non-convex or non-smooth loss functions. However, a vast majority of the current DP ERM algorithms primarily address linear ERM, suitable for datasets with linear structures.

Kernel methods hold significant importance in machine learning due to their ability to capture intricate, non-linear structures. Kernel-based learning is ubiquitous in both supervised and unsupervised settings as well as in hypothesis testing problems. For an extensive review on the subject, see Muandet et al. (2017). This work focuses on the supervised kernel learning problem via ERM as well as kernel mean embedding (KME) for probability distributions. Developing a DP framework for *general* kernels

---

[*]Corresponding authors

poses a significant hurdle due to their reliance on the local attributes of the data. For instance, not every kernel is amenable to a well-behaved DP solution. As a rather extreme case, the Kronecker delta kernel, $k(x, y) = \mathbf{1}(x = y)$, yields a meaningful solution only when the test data exactly matches one of the training instances. Consequently, the solution becomes highly sensitive to the data changes, exhibiting substantial deviations with even a single data alteration.

As a result, there has been relatively little DP research regarding kernel ERM, especially compared to linear counterparts. Chaudhuri et al. (2011) suggested transforming kernel ERM into a linear form via random features, which facilitates scaling-down operations. However, their method is limited to translation-invariant kernels, excluding a wide variety of other kernels, such as polynomial kernels or the pyramid match kernel from the computer vision domain (Grauman and Darrell, 2007), the chi-squared kernel in hand gesture recognition (Abadi et al., 2015), Histogram intersection kernel in image classification (Maji et al., 2013), Bayesian kernel in protein prediction (Alashwal et al., 2009), and the diffusion kernel in manifold learning (Lafferty and Lebanon, 2005). Hall et al. (2013) proposed perturbing the output function of kernel ERM. Although their work can be applied to general kernels, it lacks scalability in practice (e.g., $O(n^3)$ in kernel ridge regression). Additionally, it requires access to the training data for every prediction, posing a potential threat of privacy leakage. The DP kernel ERM proposed by Jain and Thakurta (2013) can work with general kernels; however, its utility is guaranteed only when test data are accessible.

KME is a pivotal tool in diverse applications such as two-sample testing and synthetic data generation (Harder et al., 2021). It enables the comparison of probability distributions via metric in Reproducing Kernel Hilbert Space (RKHS), by embedding these distributions into elements of the RKHS. Additionally, Balog et al. (2018) found utility of KME for the private release of data. The embedding itself acts as a concise representation of the distributional information of the data. However, their methods appear to suffer in high-dimensional settings. Moreover, it is important to note that the DP functional release algorithm suggested by Hall et al. (2013) cannot be directly applied to DP KME. This limitation arises because the output is not a member of the RKHS, which is a crucial distinction since statistical methods related to KME heavily depend on operations within the RKHS.

Our study is motivated by the frequent oversight in prior research of the practical challenges in real-world implementations, such as scalability, private implementation without access to test data, accurate DP KME release, and the lack of information on future private data use. We propose algorithms designed for practical DP kernel ERM by integrating the Nyström method. Although the Nyström method has been a state-of-the-art technique with random Fourier features in scalable kernel learning, this work is the first to apply it to DP kernel learning. The proposed algorithms are designed to be scalable and are suitable even for unregularized learning or non-convex loss functions. Moreover, we demonstrate that our Nyström-based techniques can be used to construct a DP estimate of KME, which can be utilized for private two-sample tests via maximal mean discrepancy, for example. We also address the cases where the model is unknown, such as in public releases of private data, by employing an algorithm that releases versatile private data suitable for kernel ERM with diverse objectives.

## 1.1 Related Works

**DP ERM.** ERM serves as a fundamental tool in machine learning, with substantial literature on DP ERM dating back to Dwork et al. (2006b). Among the current works, most of which regard linear ERM, main approaches include output and objective perturbation methods (Chaudhuri et al., 2011; Kifer et al., 2012; Jain and Thakurta, 2014; McSherry and Talwar, 2007; Gopi et al., 2022, 2023; Mangoubi and Vishnoi, 2022), and gradient perturbation methods (Bassily et al., 2014; Wang et al., 2017; Song et al., 2013). However, Iyengar et al. (2019) pointed out that most output or objective perturbation-based algorithms suffer from scalability issues in their implementation. For example, they need to directly access optimal solutions in order to release private models, thereby reducing scalability, particularly in kernel learning. Additionally, they often require strong assumptions on learning problems to guarantee privacy, such as double differentiable loss or a strongly convex regularizer, which limits the range of applicable learning scenarios (Iyengar et al., 2019). In contrast, gradient or SGD-based output perturbation methods (Wu et al., 2017; Feldman et al., 2020) are relatively unconstrained by learning restrictions or scalability concerns. While a vast amount of literature exists on linear ERM, not all DP kernel ERM algorithms can capitalize on these findings.

**DP kernel ERM and KME.** To adapt techniques from DP linear ERM to kernel ERM, one needs random features of the kernel suitable for DP learning. Random Fourier features were suggested as a

candidate for translation-invariant kernels (Chaudhuri et al., 2011), but none have been proposed for general kernels. Hall et al. (2013) and Jain and Thakurta (2013) addressed DP kernel ERM algorithms for general kernels without using random features. However, Jain and Thakurta (2013) assumes accessible test data, which may be inappropriate in certain cases. On the other hand, Hall et al. (2013) suggests a sophisticated method for releasing a private model but requires strong regularization. Moreover, both algorithms need the true solution before releasing the private model, resulting in scalability issues. The discussion on DP KME has been somewhat distant from DP kernel ERM, initially introduced by Balog et al. (2018) as a technique for private database release. Balog et al. (2018) developed private algorithms for releasing KME for both general and translation-invariant kernels. However, their methods suffer from the curse of dimensionality in both cases. Additionally, we note that the techniques in Hall et al. (2013) and Jain and Thakurta (2013) cannot be directly adapted to DP KME release.

## 1.2 Main Contributions

**Scalable DP kernel ERM for general kernels.** We develop scalable algorithms for DP kernel ERM that accommodate general kernels, particularly non-translation-invariant ones. The Nyström method serves as a principal tool that enables much-desired scalability in practical applications. Our work is the first to propose a DP $K$-means Nyström method and incorporate it in DP kernel ERM. Also, to the best of our knowledge, this is the first attempt to compare the theoretical and experimental efficacy of scalable private kernel learning algorithms. The proposed algorithms reduce time complexity and memory costs, akin to the Nyström method in non-private settings. Furthermore, experiments in Section 4 demonstrate the superior performance of our algorithms for different learning problems. We present a brief comparison of our approach and three existing methods in Table 1.

Table 1: Comparison of DP kernel ERM algorithms in terms of restrictions for privacy guarantee.

| Algorithms | General kernels | Scalable | Test data free | General objective |
|---|---|---|---|---|
| Chaudhuri et al. (2011) | ✗ | ✓ | ✓ | ✓ |
| Jain and Thakurta (2013) | ✓ | ✗ | ✗ | ✗ |
| Hall et al. (2013) | ✓ | ✗ | ✓ | ✗ |
| Proposed | ✓ | ✓ | ✓ | ✓ |

**DP KME.** We propose a data-dependent DP KME releasing algorithm based on the DP $K$-means Nyström method. The error of the DP KME is shown to be primarily related to how accurately the kernel matrix is estimated by the Nyström method. Our empirical study suggests its superiority compared to alternative DP KME releasing algorithms.

**Private data release for versatile kernel learning.** We consider the offline setting for database release, where the database owner releases data without knowledge of the models or statistics desired by users. We provide a DP algorithm that generates a dataset yielding excess empirical risk bounds of $O(n^{-c})$ for the logistic and the Linex (Ma et al., 2019) losses if it is used for kernel ERM.

## 2 Preliminaries

### 2.1 Differential Privacy

Among the numerous variants of DP, this work primarily employs $(\epsilon, \delta)$-DP, as it is widely applicable to various statistical procedures. Let $\mathcal{X}$ be the data space. We consider a randomized algorithm $M : \mathcal{X}^n \to \mathcal{P}(\mathcal{Z})$, where $\mathcal{P}(\mathcal{Z})$ is a family of distributions over an output space $\mathcal{Z}$. Let $D \sim D'$ indicate that datasets $D$ and $D'$ are neighbors, meaning they differ only by a single individual.

**Definition 2.1** (($\epsilon, \delta$)-DP, Dwork et al. (2006a))**.** An algorithm $M$ is $(\epsilon, \delta)$-DP if the following holds:
$$\sup_{D, D' \subset \mathcal{X}^n, D \sim D'} \sup_{A \subset \mathcal{Z}} \mathbb{P}\left(M(D) \in A\right) - e^{\epsilon}\mathbb{P}\left(M(D') \in A\right) \leq \delta.$$

Definition 2.1 states that an algorithm is DP if its randomness is proportional to the deviation in the output due to a single data change, as demonstrated in Proposition 1.

**Proposition 1** (Gaussian mechanism, Nikolov et al. (2013))**.** *If a deterministic algorithm $\mathcal{A} : \mathcal{X}^n \to \mathbb{R}^d$ satisfies $\sup_{D, D' \subset \mathcal{X}^n, D \sim D'} \|\mathcal{A}(D) - \mathcal{A}(D')\|_2 \leq \Delta$, an algorithm defined by $M(D) := \mathcal{A}(D) + \frac{\Delta\left(1 + \sqrt{2\log\frac{1}{\delta}}\right)}{\epsilon}\varepsilon$ is $(\epsilon, \delta)$-DP where $\varepsilon$ follows the standard normal distribution on $\mathbb{R}^d$.*

Statistical procedures, including the algorithms we propose, typically involve multiple steps, such as data-dependent evaluations and multiple references to the data. Proposition 2 provides a privacy guarantee for such procedures.

**Proposition 2** (Composition theorem, Dwork et al. (2006a)). *For algorithms $M_1 : \mathcal{X}^n \to \mathcal{P}(\mathcal{Z})$ and $M_2 : \mathcal{X}^n \times \mathcal{Z} \to \mathcal{P}(\mathcal{W})$, the algorithm $M : \mathcal{X}^n \to \mathcal{Z}$ defined by $M(D) = M_2(D, M_1(D))$ for $D \subset \mathcal{X}^n$ is $(\epsilon_1 + \epsilon_2, \delta_1 + \delta_2)$-DP if $M_1$ is $(\epsilon_1, \delta_1)$-DP and $M_2(\cdot, z)$ is $(\epsilon_2, \delta_2)$-DP for every $z \in \mathcal{Z}$.*

## 2.2 Kernel, RKHS, and Random Features

Kernel methods learn a non-linear structure by using a non-linear map, called a feature map $\phi$, to transform the data $\{x_i\}_{i=1}^n$ and analyze the linear structure of $\{\phi(x_i)\}_{i=1}^n$. An appropriate choice of $\phi$ captures the underlying complex structure of the data. The feature maps are inherently determined by the kernel function $k$, a positive definite function with a corresponding RKHS $(\mathcal{H}_k, \|\cdot\|_{\mathcal{H}_k})$, satisfying $\langle \phi(x), \phi(y) \rangle_{\mathcal{H}_k} = k(x, y)$. Within the RKHS, the outputs of the feature map behave akin to vectors in Euclidean space, allowing one to learn the non-linear structure through a linear model.

A random feature is a map $\varphi : \mathcal{X} \to \mathbb{R}^m$ that approximates the kernel function in a way that $k(x, y) \approx \langle \varphi(x), \varphi(y) \rangle$, providing a linearized version of kernel learning, as will be discussed in Sections 3.2 and 3.3. Traditionally, $k(x, y) \approx \mathbb{E}[\langle \varphi(x), \varphi(y) \rangle]$ is required for a random feature map, but we adopt a more broad terminology.

### 2.2.1 Nyström Methods

Nyström methods are low-rank kernel matrix approximation methods that facilitate scalable kernel learning. For given dataset $\{x_i\}_{i=1}^n \subset \mathcal{X}$, and a kernel matrix $\mathbf{K}$, the approximation $\hat{\mathbf{K}}$ is calculated as $\hat{\mathbf{K}} := \mathbf{K_{ZX}}^T \mathbf{K_Z}^\dagger \mathbf{K_{ZX}}$ where $\mathbf{K_Z} := [k(z_i, z_j)]_{m \times m}$, and $\mathbf{K_{ZX}} := [k(z_i, x_j)]_{m \times n}$. Here, $Z = \{z_1, \ldots, z_m\} \subset \mathcal{X}$ are some pre-chosen landmark points, and $\dagger$ represents the Moore–Penrose inverse. Then, $\hat{\mathbf{K}}$ can be interpreted as projections of data points onto the plane in the RKHS:

$$\hat{\mathbf{K}} = [\langle \text{proj}_{\mathcal{S}} \phi(x_i), \text{proj}_{\mathcal{S}} \phi(x_j) \rangle_{\mathcal{H}_k}]_{n \times n}$$

where $\mathcal{S} = \text{span}\{\phi(z_i) | 1 \leq i \leq m\} \subset \mathcal{H}_k$. Thus, the accuracy of $\hat{\mathbf{K}}$, or the accuracy of scalable kernel learning via $\hat{\mathbf{K}}$, depends on how closely the subspace $\mathcal{S}$ resembles the data $\{\phi(x_i)\}_{i=1}^n$, the corresponding elements in the RKHS of the original data. Consequently, Nyström methods operate by selecting landmark points that effectively represent the original data, as demonstrated in the past studies using data-dependent landmark points such as $K$-means centroids or subsamples (Kumar et al., 2012; Zhang et al., 2008; He and Zhang, 2018).

## 2.3 Problem Formulation

**DP kernel ERM**. For a given dataset $D = \{(x_i, y_i)\}_{i=1}^n \subset (\mathcal{X} \times \mathcal{Y})^n$, a loss function $l : \mathbb{R}^2 \to \mathbb{R}_{\geq 0}$, and a regularization parameter $\lambda > 0$, we consider the following regularized kernel ERM:

$$\arg\min_{f \in \mathcal{H}_k} \hat{L}^\lambda(f; D) \triangleq \frac{1}{n} \sum_{i=1}^n l(\langle f, \phi(x_i) \rangle_{\mathcal{H}_k}, y_i) + \frac{\lambda}{2} \|f\|_{\mathcal{H}_k}^2.$$

The minimization finds a function $f$ that best explains the data under the given loss, and the regularization parameter $\lambda$ prevents over-fitting. We measure the quality of the model by the excess empirical risk of $f$, defined by $\hat{L}^\lambda(f; D) - \min_{g \in \mathcal{H}_k} \hat{L}^\lambda(g; D)$.

**DP KME**. For a given data $\{x_i\}_{i=1}^n \subset \mathcal{X}^n$, our objective is to design a DP KME release mechanism: an $(\epsilon, \delta)$-DP mechanism $M : \mathcal{X}^n \to \mathcal{P}(\mathcal{H}_k)$ that is sufficiently close to the KME $\mu_X := \mathbb{E}[\phi(X)]$ in the RKHS norm with high probability. Since it is known that the empirical KME $\hat{\mu}_X := \frac{\phi(x_1) + \cdots + \phi(x_n)}{n}$ converges to $\mu_X$ with $O_p(n^{-\frac{1}{2}})$ (Muandet et al., 2017), our DP KME algorithm will focus on releasing $\hat{\mu}_X$.

**Versatile DP kernel ERM**. We explore a DP public dataset release tailored for kernel ERM. Unlike mechanisms developed in **DP kernel ERM**, which train models for specific loss functions $l$ and regularization parameters $\lambda$, we develop a private dataset release $(\epsilon, \delta)$-DP mechanism $M : \mathcal{X}^n \to \mathcal{Z}^n$. This mechanism enables learning a kernel ERM for post-given, possibly infinitely many $l$ functions and $\lambda$ values.

# 3 Proposed Methods

Proofs for the theorems presented in this section are deferred to Appendix 6.3.

## 3.1 DP $K$-means Nyström Approximation

This section presents a pivotal tool for achieving scalability for general DP kernel learning: DP $K$-means Nyström method. It is instrumental for our DP algorithms to make predictions using released private functions without needing access to the original data, and to be easily incorporated into the current linear DP ERM framework, unlike Hall et al. (2013).

As demonstrated by Chaudhuri et al. (2011) and Balog et al. (2018), DP kernel learning can effectively utilize the established methods of DP linear learning through random features. However, the use of random Fourier features depends on specific kernel characteristics, particularly translation-invariance, which complicates the development of random features for more general kernels. To overcome this challenge, we adopt the Nyström approximation scheme, which is suitable for constructing random features for a wider range of kernels. As discussed in Section 2.2.1, selecting landmark points that accurately capture the data structure is crucial for the Nyström method. Following the landmark selection guidelines outlined in Zhang et al. (2008), namely the $K$-means approach, we provide a detailed rationale behind this criterion to demonstrate how effectively these points represent the underlying data structure.

Define $\varphi_Z^{(Nys)} : \mathcal{X} \to \mathbb{R}^m$ as $\varphi_Z^{(Nys)} := (\langle \text{proj}_{\mathcal{S}} \phi(x), b_1 \rangle_{\mathcal{H}_k}, \ldots, \langle \text{proj}_{\mathcal{S}} \phi(x), b_m \rangle_{\mathcal{H}_k})$ where $\{b_i\}_{i=1}^m$ is an orthonormal basis of $\mathcal{S}$. Here, $\varphi^{(Nys)}$ is our random feature map. Our goal is to select suitable landmarks that minimize the approximation error of $\varphi^{(Nys)}$ for the kernel $k$ at $\{(x_i, x_j)\}_{n \times n}$, which can be expressed as follows:

$$\frac{1}{n^2} \sum_{i,j=1}^n \left( k(x_i, x_j) - \langle \varphi_Z^{(Nys)}(x_i), \varphi_Z^{(Nys)}(x_j) \rangle \right)^2 = \frac{1}{n^2} \|\mathbf{K} - \hat{\mathbf{K}}\|_F^2. \tag{1}$$

Note that the error described in Eq. (1) is also known as the Nyström approximation error in the literature on low-rank kernel approximation. The accuracy of Nyström-based kernel learning depends on the accuracy of the kernel matrix approximation: $\|\mathbf{K} - \hat{\mathbf{K}}\|_2$, which is upper bounded by $\|\mathbf{K} - \hat{\mathbf{K}}\|_F$. However, optimizing the landmark selection by minimizing LHS in Eq. (1) under DP is challenging because the problem is neither convex nor Lipschitz, making DP optimization difficult. Theorem 1 offers a comprehensive approach to identifying appropriate landmark points by solving the $K$-means problem, rather than directly addressing the complex challenge.

**Theorem 1.** *If a kernel $k$ is $c'$-Lipschitz[2] for both of its arguments, the Nyström approximation error is bounded by the quantization error of the landmark points for the original dataset:*

$$\|\mathbf{K} - \hat{\mathbf{K}}\|_F \leq 2c' \sqrt{n \sum_{i=1}^n \min_{z_j \in Z} \|x_i - z_j\|_2^2}. \tag{2}$$

Theorem 1 sheds light on the connection between low-rank kernel approximation and data clustering. Since the Nyström method is essentially an orthogonal projection of data onto $\mathcal{S}$ in RKHS and the LHS of Eq. (2) resembles the common clustering objective, the theorem essentially suggests that a proper clustering can identify a subspace $\mathcal{S}$ that is closely aligned with the data. We emphasize that the bound stated in Theorem 1 is $O(\sqrt{ne})$, where $e = \sum_{i=1}^n \min_{z_j \in Z} \|x_i - z_j\|_2^2$. The bound is tighter than the $O(ne\|\mathbf{K_Z}^{-1}\|_F)$ bound provided in Zhang et al. (2008), if the sample size $n$ and the number of landmark points $m$ are sufficiently large such that $\mathbf{K_Z}$ has small eigenvalues, even though the assumption on kernel is equivalent to the one made in Zhang et al. (2008).

In Algorithm 1, we present a DP algorithm for $K$-means-based Nyström approximation. The obtained orthonormal basis $\{b_i\}_{i=1}^n$ of $\mathcal{S}$ and their corresponding random feature map $\varphi^{(Nys)}$ can be used in the subsequent DP kernel learning. For the DP K-means step in line 2, the algorithm from the `diffprivlib` package implemented by IBM [3] can be used.

**Algorithm 1** DP $K$-means Nyström approximation.

---

**Input**: given data $\{x_1, \ldots, x_n\}$, random features of dimension $m$, cluster threshold parameter $m_0$, and privacy parameters $\epsilon, \delta$.

**Output**: a random feature mapping $\varphi^{(Nys)} : \mathcal{X} \to \mathbb{R}^m$, and $\{b_1, \ldots, b_m\} \subset \mathcal{H}_k$.

    1. $K \leftarrow \lfloor m_0 \epsilon \rfloor$.
    2. $\{z_1, \ldots, z_k\} \leftarrow$ centroids of $\epsilon/2$-DP $K$-means for $\{x_1, \ldots, x_n\}$.
    3. If $K < m$, draw $\{z_{k+1}, \ldots, z_m\}$ from some distribution $Q$.
    4. $\mathbf{U}\mathbf{\Sigma}\mathbf{U}^T \leftarrow$ the unitary diagonalization of a matrix $[k(z_i, z_j)]_{m \times m}$.
    5. $\varphi^{(Nys)} \leftarrow \varphi^{(Nys)}(x) := \frac{1}{R}\mathbf{\Sigma}^{\dagger\frac{1}{2}}\mathbf{U}^T[k(z_1, x), \ldots, k(z_m, x)]^T$,
       where $R = \sqrt{\max_{x \in \mathcal{X}} k(x, x)}$.
    6. $b_i \leftarrow \sum_{j=1}^m [\mathbf{\Sigma}^{\dagger\frac{1}{2}}\mathbf{U}^T]_{ij}\phi(z_j)$ for $i = 1, 2, \ldots, m$.
    7. **return** $\varphi^{(Nys)}, \{b_1, \ldots, b_m\}$

---

**Theorem 2.** *Algorithm 1 is $\frac{1}{2}\epsilon$-DP, and $R[\varphi^{(Nys)}(x)]_i = \langle proj_{\mathcal{S}}\phi(x), b_i \rangle_{\mathcal{H}_k}$ for all $i$ and $x \in \mathcal{X}$.*

Although the privacy remains unaffected by the threshold parameter $m_0$, it does influence the quality of the landmark points: the DP $K$-means centroids. Setting $K = m$ and omitting the thresholding step will result in small cluster sizes for large $m$, making the centroid estimates more susceptible to the noise added in the DP $K$-means step. This issue is exacerbated by a small privacy budget $\epsilon$, which further amplifies the noise. To mitigate this effect, thresholding is performed using $\lfloor m_0 \epsilon \rfloor$. After applying cluster size thresholding, additional $m - K$ landmark points are generated in line 3 from a distribution $Q$. While $Q$ can be chosen arbitrarily, we choose it as a mixture of $K$ truncated normal distributions centered at $z_i$s, incorporating data information extracted from the DP $K$-means.

### 3.2 Scalable DP Kernel ERM

Using the output of Algorithm 1, we can construct a DP kernel ERM for general kernels, as presented in Algorithm 2. Since the generation of random features is performed differentially privately, any existing DP linear ERM algorithms can be subsequently applied to achieve DP, in accordance with the composition theorem in Proposition 2. As an illustration, the method in Kifer et al. (2012), detailed in Algorithm 5 in Appendix 6.2, is adapted for general kernels in line 2 of Algorithm 2.

---

**Algorithm 2** DP kernel ERM for general kernels

---

**Input**: given data $\{x_1, \ldots, x_n\}$, integer $m$, and kernel $k$.

**Output**: $\widetilde{f} \in \mathcal{H}_k$.

    1. $\varphi^{(Nys)}, \{b_1, \ldots, b_m\} \leftarrow \epsilon/2$-DP feature map and basis, output of Algorithm 1.
    2. $u \leftarrow$ by solving $(\epsilon/2, \delta)$-DP linear ERM for data $\{(\phi^{(Nys)}(x_i), y_i)\}_{i=1}^n$:
    3. $\widetilde{f} := \sum_{i=1}^n u_i b_i$.
    4. **return** $\widetilde{f}$

---

**Theorem 3.** *Algorithm 2 is $(\epsilon, \delta)$-DP.*

Line 1 of Algorithm 2 transforms the data points to random features, and line 2 solves the DP linear ERM with respect to the newly produced data $\{\varphi^{(Nys)}(x_i)\}_{i=1}^n$. This step reduces the computational complexity, for example, from $O(n^3)$ to $O(nm^2 + nmd)$ in kernel ridge regression.

The linearization through random features introduces extra learning errors beyond the DP linear ERM error, specifically the approximation error for the given kernel, which is related to the $\|\mathbf{K} - \hat{\mathbf{K}}\|_2$, as shown in Theorem 4. The experiment results in Appendix 6.1 suggest that our DP random feature algorithm approximates the Gaussian kernel better than random Fourier features for small $m$.

Also, we note that although the privacy budgets were allocated equally for step 1 and 2, it may not be the optimal choice. The Fig. 3 demonstrates the utility can be improved in other privacy allocations. However, we fix the allocation as half since tuning the optimal allocation may involve privacy leakage. We included the detail in Appendix 6.1.

**Theorem 4.** *If the loss function is $c$-Lipschitz with respect to its first argument, the excess empirical risk of Algorithm 2 equipped with $(\frac{\epsilon}{2}, \delta)$-DP linear ERM algorithm is $\mathcal{E}_{n,m}(\beta) + \frac{c\|\mathbf{K} - \hat{\mathbf{K}}\|_2}{2\lambda n}$ with probability at least $1 - \beta$ where $\mathcal{E}_{n,m}(\beta)$ is an empirical risk bound of the DP linear ERM algorithm satisfied with probability at least $1 - \beta$ for inputs in a unit ball.*

### 3.3 DP Kernel Mean Embedding

This section introduces a DP KME method using $K$-means Nyström approximation in Algorithm 3. An important consideration in DP KME is that the released private embedding should be an element of the RKHS. Note that the noise level in the release of the coefficient vector $w$ in line 3 of the algorithm is determined by Proposition 1, ensuring accuracy proportional to the sample size $n$.

---

**Algorithm 3** DP kernel mean embedding (DP KME)

**Input**: given data $\{x_1, \dots, x_n\}$, integer $m$.
**Output**: a DP kernel mean embedding $f \in \mathcal{H}_k$
1. $\varphi^{(Nys)}, \{b_1, \dots, b_m\} \leftarrow \epsilon/2$-DP feature map and basis, output of Algorithm 1.
2. $w \leftarrow R\left(\frac{1}{n}\sum_{i=1}^n \varphi^{(Nys)}(x_i) + \frac{4\left(1 + \sqrt{2\log\frac{1}{\delta}}\right)}{n\epsilon}\varepsilon\right)$ where $\varepsilon \sim N\left(0, I_{m \times m}\right)$
3. $\widetilde{\mu}_X \leftarrow \sum_{i=1}^m w_i b_i$
4. **return** $\widetilde{\mu}_X$

---

**Theorem 5.** *Algorithm 3 is $(\epsilon, \delta)$-DP.*

The following theorem regards the difference between our DP KME and the conventional empirical KME estimate $\hat{\mu}_X$.

**Theorem 6.** *The DP KME error in the RKHS norm is given as follows:*

$$\|\widetilde{\mu}_X - \hat{\mu}_X\|_{\mathcal{H}_k} \leq \frac{\|\mathbf{K} - \hat{\mathbf{K}}\|_2}{\sqrt{n}} + \frac{2R\sqrt{2m}\left(1 + \sqrt{2\log\frac{1}{\delta}}\right)}{n\epsilon}\left(\sqrt{m} + \sqrt{2\log\frac{1}{\beta}}\right)$$

*with probability at least $1 - \beta$.*

According to Theorem 6, the accuracy of DP KME depends on the accuracy of the low-rank approximation of $\mathbf{K}$. By adaptively selecting landmark points according to the data, we can effectively reduce the approximation error. The following theorem confirms the advantage of using the data distribution in DP KME.

**Theorem 7.** *If the landmarks are selected independently of the data, the KME error of Algorithm 3 would become:*

$$\left\|\widetilde{\mu}_X^{(ind)} - \hat{\mu}_X\right\|_{\mathcal{H}_k} \leq 2R\left(\frac{\beta'}{n} + \sqrt{\frac{\beta'}{n}} + \frac{\left(\sqrt{2} + 2\sqrt{\log\frac{1}{\delta}}\right)}{n\epsilon}\left(\sqrt{m} + \sqrt{\beta'}\right)\right) + \|\mu_X - proj_{\mathcal{S}}\mu_X\|_{\mathcal{H}_k}$$

*with probability at least $1 - \beta$ where $\beta' = 2\log\frac{2}{\beta}$, and $\widetilde{\mu}_X^{(ind)}$ denotes the DP KME obtained by data-independently selected landmark points.*

Theorem 7 addresses the scenario where landmark points are selected independently of the data, as suggested by Balog et al. (2018). The error depends on how closely the distribution of the landmark points $\mathcal{S}$ resembles that of the data. Consequently, disregarding the data distribution $\mu_X$ can lead to suboptimal results in kernel learning.

### 3.4 Data Release for Versatile DP Kernel ERM

Here, we consider scenarios where sensitive data must be released without knowing which ERM model(s) will be trained on it. We propose a framework for releasing privacy-preserving datasets suitable for kernel ERMs that ensures robust accuracy with diverse loss functions. The non-interactive local DP ERM framework utilizing a polynomial approximation of the gradient of the objective is employed for the purpose. The proposed Algorithm 4 is a modified version of Algorithm 5 from Zheng et al. (2017), tailored for kernel learning. Denote $\|\mathcal{Y}\| := \max_{y \in \mathcal{Y}}|y|$.

---
**Algorithm 4** DP data release for versatile DP kernel ERM
---
**Input**: given data $\{(x_1, y_1), \ldots, (x_n, y_n)\}$, integers $m, p$, and privacy budget $(\epsilon, \delta)$.
**Output**: DP data

    1. $\varphi^{(Nys)}, \{b_1, \ldots, b_m\} \leftarrow \epsilon/2$-DP feature map and basis, output of Algorithm 1.

    2. $\mu \leftarrow \frac{\epsilon}{2 + 2\sqrt{2\log\frac{1}{\delta}}}$

    3. $\widetilde{\gamma}_i \leftarrow \left( y_i \varphi^{(Nys)}(x_i) + \frac{1}{\mu}\varepsilon_0^x, y_i \varphi^{(Nys)}(x_i) + \frac{1}{\mu}\varepsilon_1^x, \ldots, y_i \varphi^{(Nys)}(x_i) + \frac{1}{\mu}\varepsilon_{\frac{p(p+1)}{2}}^x \right)$

where $\varepsilon_0^x \sim N(0, 4\|\mathcal{Y}\|^2 I_{m \times m})$ and $\varepsilon_j^x \sim N(0, 4p(p+1)\|\mathcal{Y}\|^2 I_{m \times m})$ for $j = 1, \ldots, \frac{p(p+1)}{2}$.

    4. **return** $\{\widetilde{\gamma}_i\}_{i=1}^n$
---

**Theorem 8.** *Algorithm 4 is $(\epsilon, \delta)$-DP.*

In this versatile learning problem, we add a new constraint that the parameter space has a radius $r > 0$, which is a common restriction in private ERM contexts (Kifer et al., 2012). Then the ERM problem can be written as:

$$\min_{f \in \mathcal{S}, \|f\|_{\mathcal{H}_k} \leq r} \frac{1}{n} \sum_i l(\langle f, \phi(x_i)\rangle_{\mathcal{H}_k}, y) + \frac{\lambda}{2}\|f\|_{\mathcal{H}_k}^2. \tag{3}$$

We assume the following to ensure an efficient polynomial approximation of the loss by constraining its form. Note that the assumption accommodates a broader range of existing losses; for instance, logistic loss and smooth variants of hinge loss, compared to the assumption in Zheng et al. (2017).

**Assumption 1.** The loss function satisfies $l(\hat{y}, y) = l_0(\hat{y}y)$ for convex, $c$-Lipschitz $l_0 : \mathbb{R} \to \mathbb{R}_{\geq 0}$, which is $b$-smooth i.e., it is differentiable and satisfies $l_0(t_1) - l_0(t_2) \leq l_0'(t_2)(t_1 - t_2) + \frac{b}{2}(t_1 - t_2)^2$.

Theorem 9 states that the optimization in Eq. (3) can be solved using the output of Algorithm 4 with a theoretical utility guarantee.

**Theorem 9.** *Under Assumptions 1, for a regularization parameter $\lambda > 0$ and a pth degree polynomial $h$, there exists an algorithm $\mathcal{A}_{l,\lambda}$ that takes the output of Algorithm 4 and returns a classifier with excess empirical risk:*

$$\widetilde{O}\left( \left( \alpha^2 + c^2 + a_\infty^2 \left(\frac{rp}{\mu}\right)^{2p+1} \right) \frac{mR^2\|\mathcal{Y}\|^2 \log^2 \frac{1}{\beta}}{n\lambda\mu^2} + \alpha R\|\mathcal{Y}\|r \right)$$

*with probability $1 - \beta$, where $\alpha := \|l_0'(\hat{y}y) - h(\hat{y}y)\|_\infty$ on $|\hat{y}y| \leq r\|\mathcal{Y}\|R$, $a_\infty$ is the maximum absolute value of the coefficients of $h$, and $\widetilde{O}$ is a big O notation ignoring $\log n$ factors.*

*Remarks.* The bound in Theorem 9 relies heavily on the quality of the polynomial approximation of the gradient. For example, the Huber loss[4] used for support vector machines (SVM), gives a poor guarantee: $O\left((\log n)^{-1}\right)$ with Chebyshev polynomial approximation $h$ due to the lack of smoothness of the gradient. In contrast, smooth losses, such as the logistic loss $l(\hat{y}, y) = \log(1 + e^{-\hat{y}y})$, and the Linex loss $l(\hat{y}, y) = e^{a(1-\hat{y}y)} - a(1 - \hat{y}y) - 1$, offer faster convergence of guaranteed excess empirical loss bounds: $O(n^{-c})$ for some $c > 0$. The algorithm $\mathcal{A}_{l,\lambda}$ in Theorem 9, detailed in Algorithm 7 in Appendix 6.2, utilizes an inexact oracle gradient method as outlined in Algorithm 3 of Dvurechensky and Gasnikov (2016).

## 4  Experiments

In this section, we demonstrate our proposed DP methods for DP kernel learning with simulated and real data. Specifics on the data generation can be found in Appendix 6.1.

The first example is designed to demonstrate the benefit of scalable DP kernel ERM for general kernels. The data comes from two classes with a polynomial boundary between them, making the non-translation-invariant polynomial kernel the most preferable choice for a kernel function. We

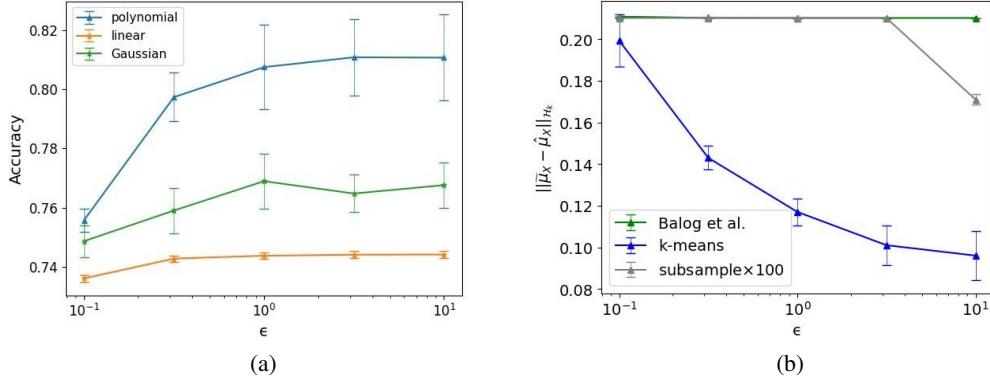

Figure 1: (a) Comparison of classification accuracy of scalable DP kernel ERMs with different kernels over a range of the privacy budget (b) Comparison of embedding errors of the proposed DP KME with alternative approaches.

consider three kernels: 3rd-order polynomial, Gaussian RBF, and linear kernels. Learning with the former two kernels is conducted as a DP linear ERM (SVM with Huber loss using the algorithm in Kifer et al. (2012)) with $m = 200$ random features obtained by Algorithm 1 and random Fourier features, respectively. The test data classification accuracy shown in Figure 1(a) clearly demonstrates the superior performance of the polynomial kernel across a wide range of the privacy budget.

We assess the estimation error of DP KME algorithms using the adult dataset (see Appendix 6.1 for details) with a Gaussian kernel for a fixed number of landmark points $m$. Although Algorithm 3 uses the $K$-means-based Nyström method, we also explore a subsampling-based version (outlined in Algorithm 6 in Appendix 6.2) with a privacy budget that is 100 times larger, as it demonstrates meaningful results in low privacy regions. We compare DP KME algorithms employing both methods, as well as Algorithm 1 in Balog et al. (2018). Figure 1(b) shows the superior performance of $K$-means-based method among the three methods. The higher errors for Balog et al. can be attributed to the misalignment of distributions between random samples and the data, causing a larger error than the error added by the noise for privacy guarantee. In contrast, Nyström methods are data-dependent, incorporating data information into their kernel approximation. However, in highly private scenarios with small $\epsilon$ values, this capability is compromised, leading to accuracy loss. Furthermore, the $K$-means-based Nyström method outperforms subsampling. This is partly due to the characteristic of the DP $K$-means algorithm, which adds noise inversely proportional to the cluster size, whereas the subsampling-based method adds noise indiscriminately. Consequently, DP $K$-means ensures that larger, more significant clusters are less perturbed, and selects important landmark points with greater accuracy, leading to improved performance in DP kernel learning. Also, it appears that the quality of the subsampling-based landmark points deteriorates easily with the addition of noise. Kernel approximation through DP subsamples is more susceptible to degradation under privacy constraints compared to DP $K$-means.

## 5   Discussion, Limitations, and Future Work

In this work, we propose a $K$-means Nyström-based private scalable kernel learning framework that is applicable for general kernels. We have developed DP kernel ERM, KME, and public data release mechanisms for versatile kernel learning within this framework. Theoretical and empirical investigations verify that our new framework is superior to existing ones, offering better performance with fewer constraints.

The Nyström method employed for the kernel matrix approximation has a few discussion points. First, we point out that utilizing DP $K$-means for scalability does not necessarily imply that the data must contain discernible clusters; instead, it helps ensure that the landmark points more effectively retain the prominent features of the data. Second, a *supervised* Nyström method, if feasible, could yield better results in certain cases. Specifically, in kernel ridge regression, focusing on a few eigenvectors aligned with the response, rather than the entire kernel matrix, might require less DP noise and enable more efficient learning.

Even though our proposed framework is designed to work with general kernels, the current implementation is limited to handling Euclidean data due to its dependence on $K$-means. A potential remedy for this limitation would be to consider a $K$-medoids type clustering method, which ensures that the centroids are selected among actual observations. Further investigation into DP $K$-medoids Nyström approximation and the development of subsequent private kernel learning methods are recommended as areas for future research.

## Acknowledgement

This work was supported by the Institute of Information & Communications Technology Planning & Evaluation (IITP) grant funded by the Korean government (MSIT) (No.2022-0-00937, Solving the problem of increasing the usability and usefulness of synthetic data algorithms for statistical data.) The work of Jeongyoun Ahn was partially supported by the National Research Foundation of Korea (NRF-2021R1A2C1093526, NRF-2022M3J6A1063021, RS-2023-00218231). The work of Cheolwoo Park was partially supported by the National Research Foundation of Korea (NRF-2021R1A2C1092925, NRF-2022M3J6A1063021).

## Footnotes

[2]A function $f$ is $c'$-Lipschitz if $|f(x) - f(y)| \leq c'|x - y|$ holds for every $x, y$.

[3]https://github.com/IBM/differential-privacy-library/blob/main/diffprivlib/models/k_means.py

[4]$l(w; x, y) := \frac{1}{2}(1.5 - yw^T x)^2 \mathbf{1}_{[0.5,1.5]}(yw^T x) + (1 - yw^T x)\mathbf{1}_{(-\infty,0.5)}(yw^T x)$ a differentiable version of the hinge loss (Chapelle, 2007)

[5]The Frobenius norm of the matrix $A = [a_{ij}]_{n \times n}$ is defined by $\|A\|_F := \sqrt{\sum_{1 \le i, j \le n} a_{ij}^2}$.

## References

Abadi, W., Fezari, M., and Hamdi, R. (2015). Bag of visualwords and chi-squared kernel support vector machine: A way to improve hand gesture recognition. In *Proceedings of the International Conference on Intelligent Information Processing, Security and Advanced Communication*, pages 1–5.

Alashwal, H., Deris, S. B., and Othman, R. M. (2009). A bayesian kernel for the prediction of protein-protein interactions. *World Academy of Science, Engineering and Technology, International Journal of Computer, Electrical, Automation, Control and Information Engineering*, 3(3):705–710.

Balog, M., Tolstikhin, I., and Schölkopf, B. (2018). Differentially private database release via kernel mean embeddings. In *Proceedings of the 35th International Conference on Machine Learning*, volume 80, pages 414–422.

Bassily, R., Feldman, V., Talwar, K., and Guha Thakurta, A. (2019). Private stochastic convex optimization with optimal rates. In *Advances in Neural Information Processing Systems*, volume 32.

Bassily, R., Smith, A., and Thakurta, A. (2014). Private empirical risk minimization: Efficient algorithms and tight error bounds. In *2014 IEEE 55th Annual Symposium on Foundations of Computer Science*, pages 464–473.

Becker, B. and Kohavi, R. (1996). Adult. UCI Machine Learning Repository. DOI: https://doi.org/10.24432/C5XW20.

Chapelle, O. (2007). Training a Support Vector Machine in the Primal. In *Large-Scale Kernel Machines*. The MIT Press.

Chaudhuri, K., Monteleoni, C., and Sarwate, A. D. (2011). Differentially private empirical risk minimization. *J. Mach. Learn. Res.*, 12(3):1069–1109.

Dong, J., Roth, A., and Su, W. J. (2022). Gaussian differential privacy. *Journal of the Royal Statistical Society Series B: Statistical Methodology*, 84(1):3–37.

Dvurechensky, P. and Gasnikov, A. (2016). Stochastic intermediate gradient method for convex problems with stochastic inexact oracle. *J Optim Theory Appl*, 171:121–145.

Dwork, C., Kenthapadi, K., McSherry, F., Mironov, I., and Naor, M. (2006a). Our data, ourselves: Privacy via distributed noise generation. In *Advances in Cryptology - EUROCRYPT 2006*, pages 486–503.

Dwork, C., McSherry, F., Nissim, K., and Smith, A. (2006b). Calibrating noise to sensitivity in private data analysis. In *Theory of Cryptography*, pages 265–284.

Feldman, V., Koren, T., and Talwar, K. (2020). Private stochastic convex optimization: optimal rates in linear time. In *Proceedings of the 52nd Annual ACM SIGACT Symposium on Theory of Computing*, page 439–449.

Gopi, S., Lee, Y. T., and Liu, D. (2022). Private convex optimization via exponential mechanism. In *Proceedings of Thirty Fifth Conference on Learning Theory*, volume 178, pages 1948–1989.

Gopi, S., Lee, Y. T., Liu, D., Shen, R., and Tian, K. (2023). Private convex optimization in general norms. In *Proceedings of the 2023 Annual ACM-SIAM Symposium on Discrete Algorithms (SODA)*, pages 5068–5089.

Grauman, K. and Darrell, T. (2007). The pyramid match kernel: Efficient learning with sets of features. *J. Mach. Learn. Res.*, 8:725–760.

Hall, R., Rinaldo, A., and Wasserman, L. (2013). Differential privacy for functions and functional data. *J. Mach. Learn. Res.*, 14(1):703–727.

Harder, F., Adamczewski, K., and Park, M. (2021). Dp-merf: Differentially private mean embeddings with randomfeatures for practical privacy-preserving data generation. In *Proceedings of The 24th International Conference on Artificial Intelligence and Statistics*, volume 130, pages 1819–1827.

He, L. and Zhang, H. (2018). Kernel k-means sampling for nyström approximation. *IEEE Transactions on Image Processing*, 27(5):2108–2120.

Iyengar, R., Near, J. P., Song, D., Thakkar, O., Thakurta, A., and Wang, L. (2019). Towards practical differentially private convex optimization. In *2019 IEEE Symposium on Security and Privacy (SP)*, pages 299–316.

Jain, P. and Thakurta, A. (2013). Differentially private learning with kernels. In *Proceedings of the 30th International Conference on Machine Learning*, volume 28, pages 118–126.

Jain, P. and Thakurta, A. G. (2014). (near) dimension independent risk bounds for differentially private learning. In *Proceedings of the 31st International Conference on Machine Learning*, volume 32, pages 476–484.

Kifer, D., Smith, A., and Thakurta, A. (2012). Private convex empirical risk minimization and high-dimensional regression. In *Proceedings of the 25th Annual Conference on Learning Theory*, volume 23, pages 25.1–25.40.

Kumar, S., Mohri, M., and Talwalkar, A. (2012). Sampling methods for the nyström method. *J. Mach. Learn. Res.*, 13:981–1006.

Lafferty, J. and Lebanon, G. (2005). Diffusion kernels on statistical manifolds. *J. Mach. Learn. Res.*, 6:129–163.

Laurent, B. and Massart, P. (2000). Adaptive estimation of a quadratic functional by model selection. *The Annals of Statistics*, 28(5):1302–1338.

Ma, Y., Zhang, Q., Li, D., and Tian, Y. (2019). Linex support vector machine for large-scale classification. *IEEE Access*, 7:70319–70331.

Maji, S., Berg, A. C., and Malik, J. (2013). Efficient classification for additive kernel svms. *IEEE Transactions on Pattern Analysis and Machine Intelligence*, 35(1):66–77.

Mangoubi, O. and Vishnoi, N. (2022). Sampling from log-concave distributions with infinity-distance guarantees. In *Advances in Neural Information Processing Systems*, volume 35, pages 12633–12646.

McSherry, F. and Talwar, K. (2007). Mechanism design via differential privacy. In *48th Annual IEEE Symposium on Foundations of Computer Science (FOCS'07)*, pages 94–103.

Moro, S., Rita, P., and Cortez, P. (2014). Bank Marketing. UCI Machine Learning Repository. DOI: https://doi.org/10.24432/C5K306.

Muandet, K., Fukumizu, K., Sriperumbudur, B., Schölkopf, B., et al. (2017). Kernel mean embedding of distributions: A review and beyond. *Foundations and Trends® in Machine Learning*, 10(1-2):1–141.

Nikolov, A., Talwar, K., and Zhang, L. (2013). The geometry of differential privacy: the sparse and approximate cases. In *Proceedings of the Forty-Fifth Annual ACM Symposium on Theory of Computing*, STOC '13, page 351–360.

Pinelis, I. (1994). Optimum bounds for the distributions of martingales in banach spaces. *The Annals of Probability*, 22(4):1679–1706.

Song, S., Chaudhuri, K., and Sarwate, A. D. (2013). Stochastic gradient descent with differentially private updates. In *2013 IEEE Global Conference on Signal and Information Processing*, pages 245–248.

Wang, D., Chen, C., and Xu, J. (2019). Differentially private empirical risk minimization with non-convex loss functions. In *Proceedings of the 36th International Conference on Machine Learning*, volume 97, pages 6526–6535. PMLR.

Wang, D., Ye, M., and Xu, J. (2017). Differentially private empirical risk minimization revisited: Faster and more general. In *Advances in Neural Information Processing Systems*, volume 30, pages 2722–2731.

Wu, X., Li, F., Kumar, A., Chaudhuri, K., Jha, S., and Naughton, J. (2017). Bolt-on differential privacy for scalable stochastic gradient descent-based analytics. In *Proceedings of the 2017 ACM International Conference on Management of Data*, SIGMOD '17, pages 1307–1322.

Yang, T., Li, Y.-f., Mahdavi, M., Jin, R., and Zhou, Z.-H. (2012). Nyström method vs random fourier features: A theoretical and empirical comparison. In *Advances in Neural Information Processing Systems*, volume 25, pages 476–484.

Zhang, K., Tsang, I. W., and Kwok, J. T. (2008). Improved nyström low-rank approximation and error analysis. In *Proceedings of the 25th International Conference on Machine Learning*, ICML '08, pages 1232–1239.

Zheng, K., Mou, W., and Wang, L. (2017). Collect at once, use effectively: Making non-interactive locally private learning possible. In *Proceedings of the 34th International Conference on Machine Learning*, volume 70, pages 4130–4139. PMLR.

# 6 Appendix

## 6.1 Experiment details and additional results

We provide specifics on the two experiments: DP kernel ERM conducted on a simulated dataset and DP KME performed on the adult dataset. We use five Intel® Xeon® Gold 6248 processors for the computing resources in the simulation and one in the other experiments.

**Privacy parameter**. Throughout the experiments, we use privacy parameters $(\epsilon, \delta) \in \{(10^{-1}, n^{-2}), (10^{-0.5}, n^{-2}), (10^{0}, n^{-2}), (10^{0.5}, n^{-2}), (10^{1}, n^{-2})\}$, where $n$ represents the dataset size used in the algorithm.

**Algorithm hyperparameters**. Throughout the experiments, the hyperparameters for Algorithm 1 are set as follows: the distribution $Q$ is defined as a mixture of $m$ truncated normal distributions, each supported on $[0,1]^d$ and centered at $\{z_i\}_{i=1^m}$, with standard deviation $\sigma_i := \max_{j \neq i} \|z_j - z_i\|_2$. Additionally, the thresholding parameter $m_0$ is set to $\lfloor 0.01n \rfloor$.

### 6.1.1 Scalable DP Kernel ERM

**Simulation dataset generation**. The feature vector of the simulated dataset is drawn independently from a mixture of truncated normal distributions supported on $[0,1]^{200}$. The following is the density formula:

$$f(x) = \frac{1}{4} \sum_{i=1}^{4} dN_t(\mu_i, 0.04I_{200 \times 200}),$$

where $dN_t(\mu, \sigma I_{200 \times 200})$ is the density of the coordinate-wise truncated normal distribution, that is:

$$(X_1, \ldots, X_{200}) \sim dN_t(\mu, \sigma^2 I_{200 \times 200}) \Leftrightarrow X_i \sim Z_i | Z_i \in [0,1] \text{ where } Z_i \sim N(\mu_i, \sigma^2).$$

The mean values of the truncated normal distribution are as follows:

$$
\begin{aligned}
\mu_1 &= \underbrace{(0.7,\ldots,0.7)}_{200} \\
\mu_2 &= \underbrace{(0,\ldots,0)}_{200} \\
\mu_3 &= \underbrace{(0.5,\ldots,0.5,}_{100}\underbrace{0,\ldots,0)}_{100} \\
\mu_4 &= \underbrace{(0,\ldots,0,}_{100}\underbrace{0.5,\ldots,0.5)}_{100}.
\end{aligned}
$$

The response variable $y$ is obtained as follows:

$$
y = \mathrm{sign}\left(\left(\left[Z\left(x - \frac{1}{2}\mathbf{1}\right)\right)^{\otimes 3}\right] w + \mathrm{noise}\right),
$$

where $\otimes$ denotes the coordinate-wise power operation, $Z$ is a $20 \times 200$ random matrix with entries drawn from the standard normal distribution, $w$ is a random vector following $N(\mathbf{0}, I_{20\times 20})$, and $\mathbf{1}$ and $\mathbf{0}$ are vectors with all components equal to 0 or 1, respectively. The noise is assigned according to the standard normal distribution. Additionally, in the DP polynomial kernel ERM, we preprocess the data to ensure their norm remains below 1, a privacy-preserving procedure. The experiment was repeated 10 times, 1,000,000 training samples and 200,000 test samples were generated for each run to solve and evaluate the accuracy of the DP kernel ERM.

**DP Kernel ERM hyperprameter**. The regularization parameter $\lambda \in \{10^{-5}, 10^{-4}, 10^{-3}, 10^{-2.5}, 10^{-2}\}$ is used in tuning. Figure 1a shows the highest accuracy achieved for each privacy parameter across these values of $\lambda$.

**Preprocessing of the data in simulation** Additionally, we constrain the $\ell_2$ norm of the data to 1 when employing both linear ERM and polynomial kernel ERM. This step is essential for DP kernel ERM, particularly when the kernel is unbounded, as it requires the boundedness of the data.

**Kernel used in DP Kernel ERM**. The Gaussian kernel $e^{-\frac{\|x-y\|_2^2}{512}}$ and the 3rd-order kernel $\left(\frac{\langle x,y\rangle + 1}{2}\right)^3$ are used. The hyperparameter $\sigma$ of the Gaussian kernel is selected by a grid search in a non-private kernel SVM setting: we generate 1000 training data and 200 test data, then compare the test error of the non-private kernel SVM for the Gaussian kernel $e^{-\frac{\|x-y\|_2^2}{2\sigma^2}}$ over $\sigma = 2^{-2}, 2^{-1}, \ldots, 2^7$. The $\sigma$ with the lowest test error is selected. Each non-private kernel SVM for the kernel is repeated 10 times, and the averaged errors are compared. The 3rd-order kernel is scaled to ensure its maximum value is at least 1 for data within a unit ball.

### 6.1.2 DP KME release experiment

**Data description**. The real dataset, the adult dataset(Becker and Kohavi, 1996), can be downloaded from the UCI Machine Learning Repository. It consists of 48,842 observations for binary classification whether the income is above 50K or less than 50K.

**Adult dataset processing**. In the data processing, we follow the conventional practice of existing works on DP ERM, for example, Chaudhuri et al. (2011); Iyengar et al. (2019). Continuous features (all positive) underwent scaling by their maximum values, while categorical features were expanded via one-hot encoding. To eliminate collinearity, we omit the last column in each of the one-hot encoded features, resulting in a dimensionality of 103.

**Kernel choice**. For the empirical KME release, we use the entire dataset to produce a DP KME based on the Gaussian kernel $k(x, y) = e^{-\frac{\|x-y\|_2^2}{2}}$, and $m = \lfloor\sqrt{n}\rfloor$ dimension of random features for comparison.

**Approximation evaluation**. The approximation error is evaluated by the RKHS distance between the empirical KME and the DP KME. The experiment is repeated 10 times, and the average error along with the standard deviation for each privacy parameter $\epsilon$ are displayed in Figure 1b.

**DP KME algorithms**. We compare Algorithm 1 from Balog et al. (2018) with our DP KME algorithms, which use DP $K$-means and subsampling-based Nyström methods. The DP $K$-means-based algorithm corresponds to Algorithm 3, while the subsampling-based algorithm is described in Algorithm 6. Algorithm 1 from Balog et al. (2018) is implemented using their code, with a single modification: the random sample distribution is changed from the normal distribution to the uniform distribution on the unit cube, as our data is preprocessed to fit within this unit cube.

### 6.1.3   Additional experiments

**Landmark quality: kernel approximation experiments.** We evaluate the performance of three DP landmark-based methods and random Fourier features-based low-rank kernel approximation, across various settings of rank $m$. The rank of the approximation corresponds to the number of the landmark points in the Nyström methods and the dimension of the random features in the random Fourier features method. Notably, even when the number of landmark points is small, random Fourier features tend to perform less effectively than Nyström-based methods.

The kernel matrices are calculated from the Gaussian kernels $k(x,y) = e^{-\frac{\|x-y\|_2^2}{2\sigma^2}}$ for $\sigma = 2^i$ with $i = 0, 1, 2, 3$ across the entire adult datasets. The approximation error is assessed by the relative error with respect to the Frobenius norm[5]: $\frac{\|\mathbf{K} - \hat{\mathbf{K}}\|_F}{\|\mathbf{K}\|_F}$. The experiments are repeated 10 times, and the averages of the relative errors along with their standard deviations are presented in Figure 2.

**Privacy allocation.** In Algorithm 2, we allocate the privacy budgets equally between the construction of DP random features and DP ERM. Figure 3 compares the accuracies of DP kernel ERM using Algorithm 2 with various privacy allocations. The blue line shows allocating 20% of the privacy budget to the DP kernel random features construction and 80% to the DP ERM achieves better accuracy than allocating them equally. A possible heuristic rule for determining the optimal privacy allocation ratio is to allocate the budget depending on the strength of the clustered structures in the data. For instance, when the number of landmarks $m$ is small relative to the sample size $n$, it is advisable to allocate a larger portion of the privacy budget to the linear ERM rather than the DP $K$-means. This is because DP $K$-means algorithms are more accurate when $m$ is small. A smaller $m$ typically results in larger cluster sizes. Since DP $K$-means algorithms acquire private centroids by averaging the members of each cluster privately, larger clusters tend to lead to more accurate centroids given fixed privacy budgets. Therefore, we can afford to allocate more privacy resources to linear ERM.

**DP KME experiments.**

In what follows, we present a comparison between Algorithm 3 and other algorithms (Algorithm 6 and Algorithm 1 in Balog et al. (2018)) using datasets other than adult datasets in Figure 4. The datasets used include bank, CDC, credit, and MNIST, which are described below. Due to the large size of these datasets, only 30,000 examples are utilized in the experiments.

**Bank.** The Bank dataset pertains to telemarketing phone calls made by a Portuguese banking institution from 2008 to 2013 (Moro et al., 2014). It includes information on clients' financial and social backgrounds, as well as contact details from the bank, with a binary label indicating whether a deposit subscription was made. This dataset is available in the UCI database (https://archive.ics.uci.edu/dataset/222/bank+marketing). It contains 45,211 examples, comprising 5,289 positive cases and 39,922 negative cases. The dataset features 6 numerical variables and 8 categorical variables, which are one-hot encoded.

**CDC.** The CDC dataset is part of the CDC's BRFSS 2015, which consists of responses collected from Americans in the CDC's annual health-related telephone survey. This dataset includes relevant information for diabetes prediction and is available in the UCI database (https://www.archive.ics.uci.edu/dataset/891/cdc+diabetes+health+indicators). It contains demographic information, health history, personal details, and diabetes diagnoses of the respondents. The dataset comprises 253,680 examples, with 218,334 negative cases and 35,346 positive cases. It includes 21 numerical features along with a binary label.

**Credt.** The Credit dataset is related to the transactions made by European credit cardholders and is available on Kaggle (https://www.kaggle.com/datasets/mlg-ulb/creditcardfraud). This

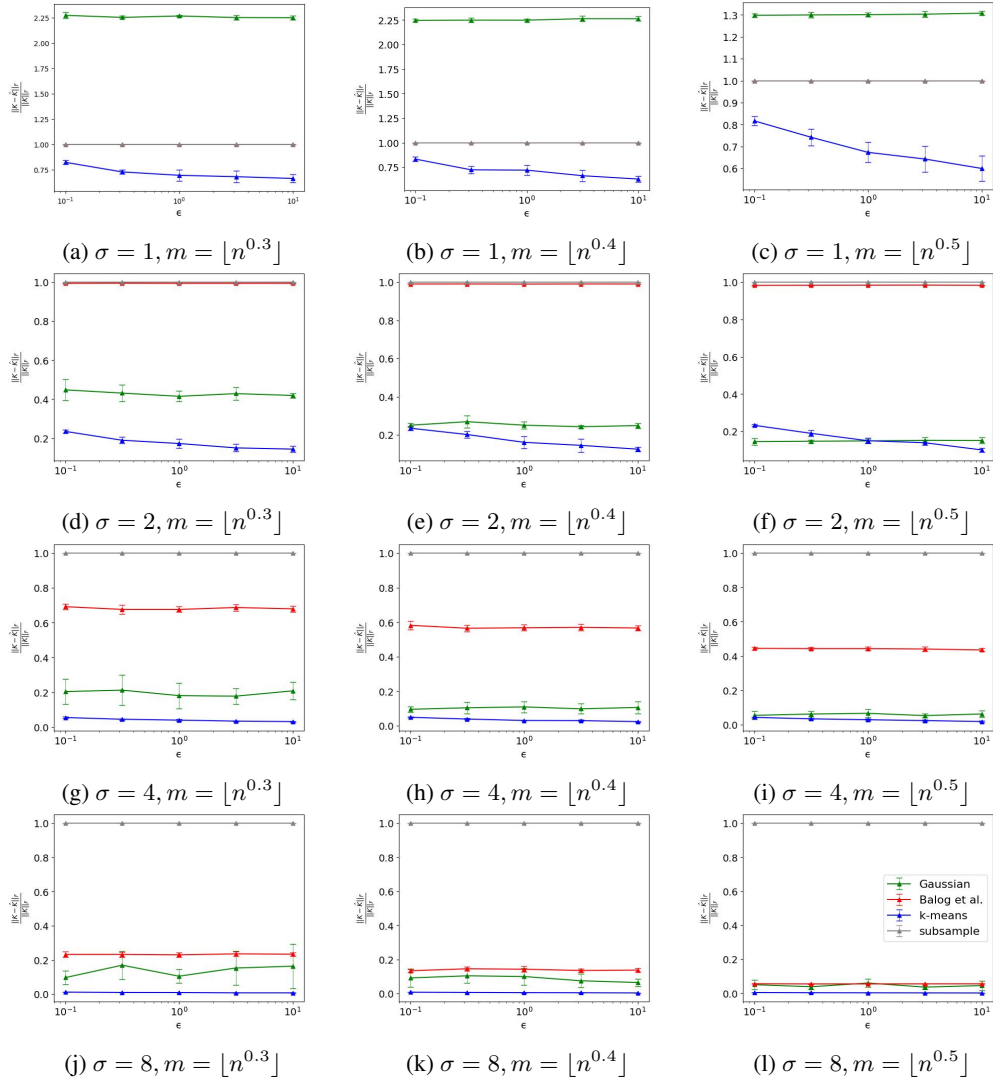

Figure 2: Comparison of four methods for various settings. In each figure, the $x$ axis represents the privacy budget, and the $y$ axis $\frac{\|\mathbf{K}-\hat{\mathbf{K}}\|_F}{\|\mathbf{K}\|_F}$.

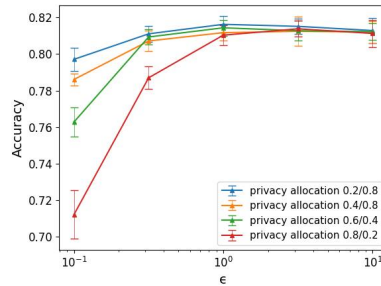

Figure 3: Classification accuracy under varying privacy budget allocations

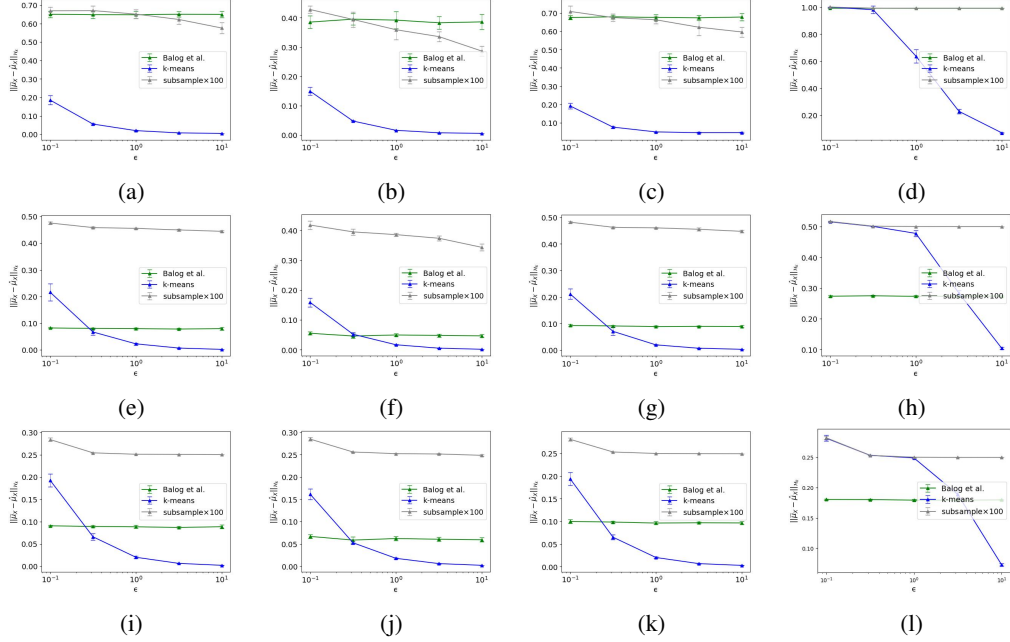

Figure 4: DP KME estimation results for Bank data (1$^{st}$ column), CDC data (2$^{nd}$ column), Credit data (3$^{rd}$ column), and MNIST data (4$^{th}$ column), using a Gaussian kernel with $\sigma = 2$ (1$^{st}$ row), and polynomial kernels of degree 2 (2$^{nd}$ row) and 4 (3$^{rd}$ row)

dataset contains 30 numerical features, two of which are 'Time' and 'Amount,' indicating the time and amount of each transaction. The remaining 28 features are principal components derived from principal components analysis due to confidentiality concerns. The dataset includes a total of 284,807 examples, with 492 labeled as fraudulent and 284,315 as non-fraudulent.

**MNIST.** The MNIST dataset contains 60,000 images of handwritten digits and can be downloaded using the PyTorch package. Each image is represented by 784 numerical features, corresponding to the pixels in a 28×28 image, along with a categorical label indicating the digits from 0 to 9.

## 6.2 Additional algorithms

Denote $\|\cdot\|_{op}$ the operator norm of a square matrix.

---

**Algorithm 5** DP linear ERM, Kifer et al. (2012)

---

**Input**: given data $\{x_1, \ldots, x_n\}$ in a unit ball, $\|\nabla_{\hat{y}} l(\hat{y}, y)\|_2 \leq \zeta$, $\|\nabla^2 l(\hat{y}, y)\|_{op} \leq \sigma$.
**Output**: $u \in \mathbb{R}^d$
     1. Set $\Delta \leq \frac{2\sigma}{\epsilon}$.
     2. Sample a random vector $b$ from $N\left(0, \frac{\zeta^2(1+\sqrt{2\log\frac{1}{\delta}})^2}{\epsilon^2} I_{m\times m}\right)$.
     3. $u \leftarrow \arg\min_{u \in \mathbb{R}^m} \frac{1}{n} \sum_{i=1}^n l(u^T x_i, y_i) + \left(\frac{\lambda}{2} + \frac{\Delta}{2n}\right) \|u\|_2^2 + \frac{b^T u}{n}$
     4. **return** $u$

---

Algorithm 6 outputs the DP Nyström-based random feature map using a subsampling Nyström method. The algorithm is $\epsilon$-DP. The proof is in the Appendix. Unlike Algorithm 1, no thresholding parameter is included since subsampling treats $m$ subsampled data equally whereas $K$-means centroids receive noise inversely to the cluster size they belong.

Algorithm 7 uses the output of Algorithm 4 to solve the kernel ERM for a given polynomial $h$ that approximates the gradient of the given loss function, and a regularization parameter $\lambda > 0$. No measure for privacy guarantee is included in Algorithm 7 since the algorithm uses only the privatized data, the output of Algorithm 4, and does not reference the original data.

---

**Algorithm 6** DP subsampling Nyström approximation.

---

**Input**: given data $\{x_1, \ldots, x_n\} \subset [0,1]^d$, random features of dimension $m$, and privacy parameter $\epsilon, \delta$.

**Output**: a random feature mapping $\varphi^{(Nys)} : \mathbb{R}^d \to \mathbb{R}^m$, and $\{b_1, \ldots, b_m\} \subset \mathcal{H}_k$.

    1. $K \leftarrow \lfloor m_0 \epsilon \rfloor$.
    2. $\{z_1^0, \ldots, z_m^0\} \leftarrow m$ uniform subsamples of $\{x_1, \ldots, x_n\}$ without replacement.
    3. $z_i \leftarrow z_i^0 + \frac{d}{\log\left(1 + \frac{n}{m}(e^{\frac{\epsilon}{2}} - 1)\right)} \varepsilon_i$ where $\varepsilon_i$ is random vector with i.i.d. components following

a Laplace distribution with density $\nu(x) = \frac{1}{2} e^{-\frac{|x|}{2}}$.

    4. $\mathbf{U\Sigma U}^T \leftarrow$ the unitary diagonalization of a matrix $[k(z_i, z_j)]_{m \times m}$.
    5. $\varphi^{(Nys)} \leftarrow \varphi^{(Nys)}(x) := \frac{1}{R} \mathbf{\Sigma}^{\dagger \frac{1}{2}} \mathbf{U}^T [k(z_1, x), \ldots, k(z_m, x)]^T$.
    6. $b_i \leftarrow \sum_{j=1}^m [\mathbf{\Sigma}^{\dagger \frac{1}{2}} \mathbf{U}^T]_{ij} \phi(z_j)$ for $i = 1, 2, \ldots, m$.
    7. **return** $\varphi^{(Nys)}, \{b_1, \ldots, b_m\}$

---

---

**Algorithm 7** Versatile DP kernel ERM

---

**Input**: given output $\{\widetilde{z}_i\}_{i=1}^n$ of Algorithm 4, integer $m$, $h$

**Output**: DP solution $\widetilde{f}$

    1. **for** $s = 0, \ldots, n-1$ **do**
    2. $\widetilde{G}(w_s; \widetilde{z}_i) \leftarrow \left( \sum_{k=0}^p a_k \prod_{j=1}^k w_s^T \widetilde{z}_{i, \frac{k(k-1)}{2} + j} \right) \widetilde{z}_{i0} + \lambda w_s$
    3. Apply Algorithm 3 in Dvurechensky and Gasnikov (2016) using $\hat{G}$ as the inexact gradient.
    4. **end for**
    5. **return** $\widetilde{f} = \sum_{i=1}^m [w_n]_i b_i$

---

## 6.3 Proofs of theorems

### 6.3.1 Pinelis's inequality

According to Theorem 3.1 in Pinelis (1994),

$$\mathbb{P}\left( \sup_j \|f_j\|_{\mathcal{H}_k} \geq r \right) \leq 2 e^{-\lambda r + \left\| \sum_{j=1}^\infty \mathbb{E}_{j-1} \left[ e^{\lambda \|d_j\|_{\mathcal{H}_k}} - 1 - \lambda \|d_j\|_{\mathcal{H}_k} \right] \right\|_\infty} \tag{4}$$

for any $\lambda > 0$, where $\{f_j\}_{j=1}^\infty$ is a martingale on $\mathcal{H}_k$ and $d_j = f_j - f_{j-1}$ with $d_0 = 0$. Let $f_j = \sum_{i=1}^j \frac{1}{n}(\phi(x_i) - \mu_X - \text{proj}_{\mathcal{S}}(\phi(x_i) - \mu_X))$ if $j \leq n$ and $f_j = f_n$ for $j > n$. Then, $f_j$ is a martingale on $\mathcal{H}_k$ so according to Eq. (4), we have

$$\mathbb{P}\left( \left\| \sum_{i=1}^j \frac{1}{n}(\phi(x_i) - \mu_X - \text{proj}_{\mathcal{S}}(\phi(x_i) - \mu_X)) \right\|_{\mathcal{H}_k} \geq r \right) \leq \mathbb{P}\left( \sup_j \|f_j\|_{\mathcal{H}_k} \geq r \right)$$

$$\leq 2 e^{-\lambda r + n\left( e^{\frac{2R}{n}\lambda} - 1 - \frac{2R}{n}\lambda \right)}$$

since $\|\phi(x_i) - \mu_X\|_{\mathcal{H}_k} \leq 2R$ and a function $e^x - 1 - x$ increases on $(0, \infty)$. Setting $\lambda = \frac{n}{2R} \log\left(1 + \frac{r}{2R}\right)$, we have

$$\mathbb{P}\left( \left\| \sum_{i=1}^j w_i(\phi(x_i) - \mu_X - \text{proj}_{\mathcal{S}}(\phi(x_i) - \mu_X)) \right\|_{\mathcal{H}_k} \geq r \right) \leq e^{\frac{nr}{2R} - n\left(\frac{r}{2R}+1\right)\log\left(1 + \frac{r}{2R}\right)}.$$

Setting $r = 2Rt$, we have

$$\mathbb{P}\left( \left\| \sum_{i=1}^j w_i(\phi(x_i) - \mu_X - \text{proj}_{\mathcal{S}}(\phi(x_i) - \mu_X)) \right\|_{\mathcal{H}_k} \geq 2Rt \right) \leq e^{-nt - n(1+t)\log(1+t)} \leq e^{-\frac{nt^2}{2(1+t)}}$$

since $\log \frac{1}{1+t} = -\frac{t}{1+t} - \frac{1}{2}\left(\frac{t}{1+t}\right)^2 - \cdots \leq -\frac{t}{1+t} - \frac{1}{2}\left(\frac{t}{1+t}\right)^2$. Putting $t = \frac{1}{n}\log\frac{1}{\beta} + \sqrt{\frac{2}{n}\log\frac{1}{\beta} + \frac{1}{n^2}\log^2\frac{1}{\beta}}$, we have

$$\left\|\sum_{i=1}^{j} w_i(\phi(x_i) - \mu_X - \text{proj}_S(\phi(x_i) - \mu_X))\right\|_{\mathcal{H}_k} \geq 2R\left(\frac{1}{n}\log\frac{1}{\beta} + \sqrt{\frac{2}{n}\log\frac{1}{\beta} + \frac{1}{n^2}\log^2\frac{1}{\beta}}\right)$$

with probability at most $\beta$. Therefore,

$$\mathbb{P}\left(\left\|\sum_{i=1}^{j} w_i(\phi(x_i) - \mu_X - \text{proj}_S(\phi(x_i) - \mu_X))\right\|_{\mathcal{H}_k} \geq 2R\left(\frac{2}{n}\log\frac{1}{\beta} + \sqrt{\frac{2}{n}\log\frac{1}{\beta}}\right)\right) \leq \beta$$

**Lemma 1** (Lemma 1 in Laurent and Massart (2000)). $\mathbb{P}\left(\|\varepsilon\|_2 \leq \sqrt{n} + \sqrt{2\log\frac{1}{\beta}}\right) \geq 1 - \beta$ *for* $\beta \in (0,1]$ *where $\varepsilon$ is a $n$ diemnsional standard normal distribution.*

### 6.3.2 Proof of Theorem 1

The Cauchy-Swartz inequality gives

$$\begin{aligned}
\frac{1}{n^2}\|\mathbf{K} - \hat{\mathbf{K}}\|_F^2 &= \frac{1}{n^2}\sum_{i,j=1}^{n}\left(k(x_i,x_j) - \langle\varphi_Z^{(Nys)}(x_i), \varphi_Z^{(Nys)}(x_j)\rangle_{\mathcal{H}_k}\right)^2 \\
&= \frac{1}{n^2}\sum_{i,j=1}^{n}\left(\langle\phi(x_i) - \text{proj}_S\phi(x_i), \phi(x_j) - \text{proj}_S\phi(x_j)\rangle_{\mathcal{H}_k}\right)^2 \\
&\leq \frac{1}{n^2}\sum_{i,j=1}^{n}\|\phi(x_i) - \text{proj}_S\phi(x_i)\|_{\mathcal{H}_k}^2\|\phi(x_j) - \text{proj}_S\phi(x_j)\|_{\mathcal{H}_k}^2 \\
&= \left(\frac{1}{n}\sum_{i=1}^{n}\|\phi(x_i) - \text{proj}_S\phi(x_i)\|_{\mathcal{H}_k}^2\right)^2 \\
&\leq \frac{1}{n}\sum_{i=1}^{n}\|\phi(x_i) - \text{proj}_S\phi(x_i)\|_{\mathcal{H}_k}^4.
\end{aligned}$$

Also, the Lipschitzness of $k$ gives

$$\begin{aligned}
\|\phi(x_i) - \text{proj}_S\phi(x_i)\|_{\mathcal{H}_k}^2 &\leq k(x_i,x_i) - \frac{k(x_i,z_j)^2}{k(z_j,z_j)} \\
&\leq k(x_i,z_j) + c'\|x_i - z_j\|_2 - \frac{k(x_i,z_j)^2}{k(x_i,z_j) + c'\|x_i - z_j\|_2} \\
&\leq \frac{(k(x_i,z_j) + c'\|x_i - z_j\|_2)^2 - k(x_i,z_j)^2}{k(x_i,z_j) + c'\|x_i - z_j\|_2} \\
&\leq c'\|x_i - z_j\|_2 \frac{2k(x_i,z_j) + c'\|x_i - z_j\|_2}{k(x_i,z_j) + c'\|x_i - z_j\|_2} \\
&\leq 2c'\|x_i - z_j\|_2
\end{aligned}$$

for every $z_j \in Z$. Thus

$$\|\mathbf{K} - \hat{\mathbf{K}}\|_F^2 \leq 2c'\sqrt{n\sum_{i=1}^{n}\|x_i - z_j\|_2^2}.$$

Choosing $z_j$s to be the $K$-means centroids, we obtain the desired.

### 6.3.3 Proof of Theorem 2

The post-processing property of the differential privacy guarantees privacy since the landmark points $z_1, \ldots, z_m$ were obtained differentially privately.

Additionally, we show $\{b_i\}_{i=1}^m$ is an orthonormal basis of $\mathcal{S}$, and $R[\varphi^{(Nys)}] = \langle \text{proj}_{\mathcal{S}} \phi(x), b_i \rangle_{\mathcal{H}_k}$.

$$
\begin{aligned}
\langle b_i, b_j \rangle_{\mathcal{H}_k} &= \sum_{k,l} [\mathbf{\Sigma}^{\dagger\frac{1}{2}} \mathbf{U}^T]_{ik} \langle \phi(z_k), \phi(z_l) \rangle_{\mathcal{H}_k} [\mathbf{\Sigma}^{\dagger\frac{1}{2}} \mathbf{U}^T]_{jl} \\
&= \sum_{k,l} [\mathbf{\Sigma}^{\dagger\frac{1}{2}} \mathbf{U}^T]_{ik} [\mathbf{K_Z}]_{kl} [\mathbf{U}\mathbf{\Sigma}^{\dagger\frac{1}{2}}]_{lj} \\
&= [\mathbf{\Sigma}^{\dagger\frac{1}{2}} \mathbf{U}^T \mathbf{K_Z} \mathbf{U} \mathbf{\Sigma}^{\dagger\frac{1}{2}}]_{ij} \\
&= [\mathbf{\Sigma}^{\dagger\frac{1}{2}} \mathbf{\Sigma} \mathbf{\Sigma}^{\dagger\frac{1}{2}}]_{ij} \\
&= \mathbf{1}(i = j \leq t_0)
\end{aligned}
$$

where $t_0$ is the rank of $\mathbf{\Sigma}$ or equivalently the rank of $\mathbf{K_Z}$. Thus the members of $\{b_i\}_{i=1}^m$ are orthonormal to each other or are zero. Ignoring the zero terms we get the orthonormal subset of $\mathcal{S}$. Also, the dimension of the spanned subspace $\mathcal{S}$ coincides to the rank of $\mathbf{K_Z}$. Thus the subset is also a basis.

Finally, $R[\varphi^{(Nys)}] = \langle \text{proj}_{\mathcal{S}} \phi(x), b_i \rangle_{\mathcal{H}_k}$ follows from the direct calculation:

$$
\begin{aligned}
\langle \text{proj}_{\mathcal{S}} \phi(x), b_i \rangle_{\mathcal{H}_k} &:= \langle \phi(x), b_i \rangle_{\mathcal{H}_k} \\
&= \langle \phi(x), \sum_{j=1}^m [\mathbf{\Sigma}^{\dagger\frac{1}{2}} \mathbf{U}^T]_{ij} \phi(z_j) \rangle_{\mathcal{H}_k} \\
&= \sum_{j=1}^m [\mathbf{\Sigma}^{\dagger\frac{1}{2}} \mathbf{U}^T]_{ij} \langle \phi(x), \phi(z_j) \rangle_{\mathcal{H}_k} \\
&= \sum_{j=1}^m [\mathbf{\Sigma}^{\dagger\frac{1}{2}} \mathbf{U}^T]_{ij} k(z_j, x) \\
&= R\varphi^{(Nys)}(x).
\end{aligned}
$$

### 6.3.4 Proof of Theorem 3

The algorithm is a composition of two DP algorithms so the $(\epsilon, \delta)$-DP is guaranteed by Proposition 2.

### Proof of Theorem 4

*Proof.* Denote the DP linear algorithm as $M$ then the excess empirical risk is:

$$
\begin{aligned}
&\hat{L}^\lambda \left( \sum_i [M(\varphi^{(Nys)}(x_1), \ldots, \varphi^{(Nys)}(x_n))]_i b_i \right) - \min_{f \in \mathcal{H}_k} \hat{L}^\lambda(f) \\
&= \underbrace{\hat{L}^\lambda(M(\phi(x_1), \ldots, \phi(x_n))) - \min_{f \in \mathcal{S}} \hat{L}^\lambda(f)}_{\text{privacy error}} + \underbrace{\min_{f \in \mathcal{S}} \hat{L}^\lambda(f) - \min_{f \in \mathcal{H}_k} \hat{L}^\lambda(f)}_{\text{random feature error}}
\end{aligned}
$$

where $b_i$s are the output of Algorithm 1, the basis of $\mathcal{S}$, and $[M(\varphi^{(Nys)}(x_1), \ldots, \varphi^{(Nys)}(x_n))]_i$ denotes the $i$th component of the output of the DP linear algorithm from input $\{\varphi^{(Nys)}(x_i)\}_{i=1}^n$.

The random feature error is bounded by

$$
\min_{f \in \mathcal{S}} \hat{L}^\lambda(f) - \min_{f \in \mathcal{H}_k} \hat{L}^\lambda(f) \leq \frac{L \|\mathbf{K} - \hat{\mathbf{K}}\|_2}{2\lambda n}
$$

by Lemma 1 in the proof of Theorem 2 in Yang et al. (2012). Then,

$$
\hat{L}^\lambda \left( \sum_i [M(\varphi^{(Nys)}(x_1), \ldots, \varphi^{(Nys)}(x_n))]_i b_i \right) - \min_{f \in \mathcal{H}_k} \hat{L}^\lambda(f) \leq \mathcal{E}_{n,m}(\beta) + \frac{L \|\mathbf{K} - \hat{\mathbf{K}}\|_2}{2\lambda n}
$$

with probability $1 - \beta$. $\qquad\square$

### 6.3.5 Proof of Theorem 5

*Proof.* We show releasing $w$ in line 2 is $(\epsilon/2, \delta)$-DP. Then Proposition 2 gaurantees the overall algorithm satisfies $(\epsilon, \delta)$-DP.

The $\ell_2$ norm of the difference between outputs of deterministic algorithm releasing $\frac{1}{n} \sum R\varphi^{(Nys)}(x_i)$ for neighboring datasets is $\frac{2R}{n}$. Applying Proposition 1 guarantees that releasing $w$ is $(\epsilon/2, \delta)$-DP. $\quad\square$

### 6.3.6 Proof of Theorem 6

*Proof.* Denote $\hat{\mu}_X^{(Nys)} := \frac{1}{n} \sum_{i=1}^{n} \sum_{j=1}^{m} R[\varphi^{(Nys)}(x_i)]_j b_j$. Since $R \sum_{j=1}^{m} R[\varphi^{(Nys)}(x_i)]_j b_j = \text{proj}_{\mathcal{S}} \phi(x_i)$,

$$
\begin{aligned}
\left\| \hat{\mu}_X - \hat{\mu}_X^{(Nys)} \right\|_{\mathcal{H}_k} &= \frac{1}{n} \left\| \sum_{i=1}^{n} \phi(x_i) - \text{proj}_{\mathcal{S}} \phi(x_i) \right\|_{\mathcal{H}_k} \\
&= \frac{\sqrt{\mathbf{1}^T (\mathbf{K} - \hat{\mathbf{K}}) \mathbf{1}}}{n} \\
&\leq \frac{\|\mathbf{K} - \hat{\mathbf{K}}\|_2}{\sqrt{n}}.
\end{aligned}
$$

Also,

$$
\left\| \widetilde{\mu}_X - \hat{\mu}_X^{(Nys)} \right\|_{\mathcal{H}_k} = \left\| \frac{1}{n} \sum_{i=1}^{n} \varphi^{(Nys)}(x_i) - w \right\|_2 = \frac{2R\left(1 + \sqrt{2\log\frac{1}{\delta}}\right)}{n\epsilon} \|\varepsilon\|_2,
$$

and from Lemma 1, we have $\mathbb{P}\left( \|\varepsilon\|_2 \leq \sqrt{m} + \sqrt{2\log\frac{1}{\beta}} \right) \geq 1 - \beta$. Therefore,

$$
\left\| \widetilde{\mu}_X - \hat{\mu}_X^{(Nys)} \right\|_{\mathcal{H}_k} \leq \frac{2R\left(1 + \sqrt{2\log\frac{1}{\delta}}\right)}{n\epsilon} \left( \sqrt{m} + \sqrt{2\log\frac{1}{\beta}} \right)
$$

with probability $1 - \beta$. $\quad\square$

### 6.3.7 Proof of Theorem 7

*Proof.* Denote $\hat{\mu}_X^{(\text{ind})} := \frac{1}{n} \sum_{i=1}^{n} \sum_{j=1}^{m} R[\varphi^{(i)}(x_i)]_j b_j$ where $\varphi^{(i)}$ is a random feature map from Algorithm 1 replacing $\{z_i\}_{i=1}^{k}$ to data independent landmark. Since $R \sum_{j=1}^{m} R[\varphi^{(i)}(x_i)]_j b_j = \text{proj}_{\mathcal{S}} \phi(x_i)$,

$$
\left\| \hat{\mu}_X^{(\text{ind})} - \hat{\mu}_X \right\|_{\mathcal{H}_k} = \frac{1}{n} \left\| \sum_{i=1}^{n} \phi(x_i) - \text{proj}_{\mathcal{S}} \phi(x_i) \right\|_{\mathcal{H}_k}.
$$

The Pinelis inequality gives:

$$
\mathbb{P}\left( \left\| \sum_{i=1}^{j} \frac{1}{n} (\phi(x_i) - \mu_X - \text{proj}_{\mathcal{S}}(\phi(x_i) - \mu_X)) \right\|_{\mathcal{H}_k} \geq 2R\left( \frac{2}{n}\log\frac{1}{\beta} + \sqrt{\frac{2}{n}\log\frac{1}{\beta}} \right) \right) \leq \beta.
$$

Thus,

$$
\mathbb{P}\left( \left\| \hat{\mu}_X^{(\text{ind})} - \hat{\mu}_X \right\|_{\mathcal{H}_k} \leq 2R\left( \frac{2}{n}\log\frac{1}{\beta} + \sqrt{\frac{2}{n}\log\frac{1}{\beta}} \right) + \|\mu_X - \text{proj}_{\mathcal{S}}\mu_X\|_{\mathcal{H}_k} \right) \geq 1 - \beta.
$$

Also,

$$
\left\| \widetilde{\mu}_X^{(\text{ind})} - \hat{\mu}_X^{(\text{ind})} \right\|_{\mathcal{H}_k} = \frac{2R\left(1 + \sqrt{2\log\frac{1}{\delta}}\right)}{n\epsilon} \|\varepsilon\|_2,
$$

and from Lemma 1, we have $\mathbb{P}\left( \|\varepsilon\|_2 \leq \sqrt{m} + \sqrt{2\log\frac{1}{\beta}} \right) \geq 1 - \beta$. Therefore,

$$
\left\| \widetilde{\mu}_X^{(\text{ind})} - \hat{\mu}_X \right\|_{\mathcal{H}_k} \leq 2R\left( \frac{2}{n}\log\frac{1}{\beta} + \sqrt{\frac{2}{n}\log\frac{1}{\beta}} + \text{err}_{priv} \right) + \|\mu_X - \text{proj}_{\mathcal{S}}\mu_X\|_{\mathcal{H}_k}
$$

with probability $1 - 2\beta$ where $\text{err}_{priv} = \frac{\left(\sqrt{2} + 2\sqrt{\log\frac{1}{\delta}}\right)}{n\epsilon} \left( \sqrt{m} + \sqrt{2\log\frac{1}{\beta}} \right)$. $\quad\square$

### 6.3.8 Proof of Theorem 8

We show line 3 is $(\epsilon/2, \delta)$-DP. Then the Proposition 2 guarantees the overall algorithm satisfies $(\epsilon, \delta)$-DP.

**Proposition 3** (Composition theorem for Gaussian noise addition, Dong et al. (2022)). *For $\mu_1, \mu_2 > 0$, and deterministic algorithms $A_1 : \mathcal{X}^n \to \mathcal{Z}$ and $A_2 : \mathcal{X}^n \times \mathcal{Z} \to \mathcal{Z}$, define algorithms $M_1 : \mathcal{X}^n \to \mathcal{P}(\mathcal{Z})$ and $M_2 : \mathcal{X}^n \times \mathcal{Z} \to \mathcal{P}(\mathcal{Z})$ as*

$$
\begin{aligned}
M_1(D) &= A_1(D) + \frac{\Delta_1}{\mu_1}\varepsilon_1 \\
M_2(D, z) &= A_2(D, z) + \frac{\Delta_2}{\mu_2}\varepsilon_2
\end{aligned}
$$

*where $\Delta_1 = \sup_{D \sim D'}\|A_1(D) - A_1(D')\|_2$, and $\Delta_2 = \sup_{z \in \mathcal{Z}}\sup_{D \sim D'}\|A_1(D, z) - A_1(D', z)\|_2$. Then $M_3 : \mathcal{X}^n \to \mathcal{P}(\mathcal{Z})$ defined by $M_3(D) = M_2(D, M_1(D))$ is $(\epsilon, \delta)$-DP if $\sqrt{\mu_1^2 + \mu_2^2} = \frac{1 + \sqrt{2\log\frac{1}{\delta}}}{\epsilon}$.*

The $\ell_2$ norm of the difference between outputs of deterministic algorithm releasing $\left(y_1\varphi^{(Nys)}(x_1), \ldots, y_n\varphi^{(Nys)}(x_n)\right)$ for neighboring datasets is at most $2\|\mathcal{Y}\|$ since $\|y\varphi^{(Nys)}\| \leq \|\mathcal{Y}\|$. Thus if we apply Proposition 1 repeatedly to algorithms

$$
\begin{aligned}
M_0(D) &:= \left(y_1\varphi^{(Nys)}(x_1) + \frac{\sqrt{2}}{\mu}\varepsilon_1^0, \ldots, y_n\varphi^{(Nys)}(x_n) + \frac{\sqrt{2}}{\mu}\varepsilon_n^0\right) \\
M_i(D) &:= \left(y_1\varphi^{(Nys)}(x_1) + \frac{p(p+1)}{\mu}\varepsilon_1^i, \ldots, y_n\varphi^{(Nys)}(x_n) + \frac{p(p+1)}{\mu}\varepsilon_n^i\right)
\end{aligned}
$$

where $\varepsilon_i^j$ are normal random vector with mean 0 and covariance $4\|\mathcal{Y}\|^2 I_{m \times m}$, releasing the outputs of $M_0, M_1, \ldots, M_{\frac{p(p+1)}{2}}$ at once is $(\epsilon/2, \delta)$-DP since $\mu = \frac{2 + 2\sqrt{2\log\frac{1}{\delta}}}{\epsilon}$. Therefore the overall algorithm satisfies $(\epsilon, \delta)$-DP.

### 6.3.9 Proof of Theorem 9

Denote $h(\hat{y}y) = \sum_{k=0}^{p} a_k(y\hat{y})^k$. We show that Algorithm 7 satisfies the theorem.

**Definition 6.1** (Dvurechensky and Gasnikov (2016), modified). *A convex function $f : Q \subset \mathbb{R}^d \to \mathbb{R}$ is endowed with a $(\delta, b, \sigma)$ stochastic oracle if an inexact gradient function $\widetilde{g} : Q \to \mathbb{R}^d$ and inexact function $f_0 : Q \to \mathbb{R}$ were given such that*

$$
\begin{aligned}
0 \leq f(y) - f_0(x) - \mathbb{E}\left[\widetilde{g}(x)\right]^T (y - x) &\leq \frac{b}{2}\|y - x\|_2^2 + \delta \\
\mathbb{E}\left[\|\widetilde{g} - \mathbb{E}\left[\widetilde{g}(x)\right]\|_2^2\right] &\leq \sigma^2.
\end{aligned}
$$

**Lemma 2** (Dvurechensky and Gasnikov (2016)). *For a $\lambda$-strongly convex function $f$ endowed with a $(\delta, b, \sigma)$ stochastic oracle, the sequecne $\{w_i\}_{i=1}^n$ generated by the algorithm 3 in Dvurechensky and Gasnikov (2016):*

$$
f(w_n) - \argmin_{\|w\|_2 \leq r} f(w) \geq O\left(\frac{\sigma^2\left(1 + \log\frac{\lambda^2 r^2 n}{\beta\sigma^2}\right)^2}{2n\lambda} + \delta^2\right)
$$

*with probability at least $1 - \beta$.*

*Proof.* Set $Q = \{x \in \mathbb{R}^m : |x| \leq r\}$. Then, the projection mapping $\varphi^{(Nys)} : \mathcal{X} \to \mathbb{R}^m$ maps $x \in \mathcal{X}$ to $\frac{1}{R}(\langle \text{proj}_{\mathcal{S}}\phi(x), b_1\rangle_{\mathcal{H}_k}, \ldots, \langle \text{proj}_{\mathcal{S}}\phi(x), b_m\rangle_{\mathcal{H}_k})$ where $\{b_i\}_{i=1}^m$ is a orthonormal basis of $S$ transform the kernel ERM to linear ERM. Denote $w = (\langle \text{proj}_{\mathcal{S}}f, b_1\rangle_{\mathcal{H}_k}, \ldots, \langle \text{proj}_{\mathcal{S}}f, b_m\rangle_{\mathcal{H}_k})$ then constraint $\|f\|_{\mathcal{H}_k} \leq r$ is equivalent to $\|w\|_2 \leq r$ since $f \in \mathcal{S}$. Then, the transformed linear ERM is

$$
\min_{\|w\|_2 \leq r} \frac{1}{n}\sum_{i=1}^n l(w^T\varphi^{(Nys)}(x_i), y_i) + \frac{\lambda}{2}\|w\|_2^2.
$$

For given $x, y$, denote $j(w; x, y) := l(w^T \varphi^{(Nys)}(x), y) + \frac{\lambda}{2}\|w\|_2^2$. Then,

$$\nabla_w l(w^T \varphi^{(Nys)}(x), y) = l_0'(w^T y \varphi^{(Nys)}(x)) y \varphi^{(Nys)}(x).$$

Denote

$$
\begin{aligned}
g(w, x, y) &:= h(y w^T \varphi^{(Nys)}(x)) \\
\widetilde{g}(w, x, y) &:= \sum_{k=0}^{p} a_k \prod_{j=1}^{k} w^T \widetilde{z}_{\frac{k(k-1)}{2}+j}
\end{aligned}
$$

where $\widetilde{z} = \left( yx + \frac{\sqrt{2}}{\mu}\varepsilon_0^x, yx + \frac{\sqrt{p(p+1)}}{\mu}\varepsilon_1^x, \cdots, yx + \frac{\sqrt{p(p+1)}}{\mu}\varepsilon_{\frac{p(p+1)}{2}}^x \right)$ following the notation in Algorithm 4. Then

$$j(w_1; x, y) - j(w_2; x, y) - (g(w_2, x, y) y \varphi^{(Nys)}(x) + \lambda w_2)^T (w_1 - w_2)$$

$$\leq (l_0'(w_2^T y \varphi^{(Nys)}(x)) - g) y \varphi^{(Nys)}(x)^T (w_1 - w_2) + \frac{\lambda + bR^2\|\mathcal{Y}\|^2}{2}\|w_1 - w_2\|_2^2$$

$$\leq 2R\|\mathcal{Y}\|\alpha r + \frac{\lambda + bR^2\|\mathcal{Y}\|^2}{2}\|w_1 - w_2\|_2^2$$

and similarly

$$j(w_1; x, y) - j(w_2; x, y) - (g(w_2, x, y)\varphi^{(Nys)}(x) + \lambda w_2)^T (w_1 - w_2) \geq -2R\|\mathcal{Y}\|\alpha r.$$

Next, we evaluate the variance of the gradient estimator:

$$\mathbb{E}\left[ \|g(w, x_i, y_i) y_i \varphi^{(Nys)}(x_i) + \lambda w - \widetilde{g}(w, x_i, y_i)\widetilde{z}_i - \lambda w\|_2^2 \right]$$

$$\leq 2\mathbb{E}\left[ \|gy\varphi^{(Nys)}(x_i) - \widetilde{g}\widetilde{z}_{i0}\|_2^2 \right] + 2\mathbb{E}\left[ \|\widetilde{g} y_i \varphi^{(Nys)}(x_i) - \widetilde{g}\widetilde{z}_{i0}\|_2^2 \right]$$

where $g, \widetilde{g}$ stands for $g(w, x_i, y_i), \widetilde{g}(w, x_i, y_i)$. The former can be bounded as

$$
\begin{aligned}
\mathbb{E}\left[ \|gy_i\varphi^{(Nys)}(x_i) - \widetilde{g}y_i\varphi^{(Nys)}(x_i)\|_2^2 \right] &= y_i^2 \|\varphi^{(Nys)}(x_i)\|_2^2 \mathrm{Var}(\widetilde{g}) \\
&\leq \|\mathcal{Y}\|^2 R^2 \sum_{k=0}^{p} \mathrm{Var}\left( a_k \prod_{j=1}^{k} w^T \widetilde{z}_{i, \frac{k(k-1)}{2}+j} \right) \\
&\leq \|\mathcal{Y}\|^2 R^2 \sum_{k=0}^{p} \mathbb{E}\left[ \left( a_k \prod_{j=1}^{k} w^T \widetilde{z}_{i, \frac{k(k-1)}{2}+j} \right)^2 \right] \\
&= \|\mathcal{Y}\|^2 R^2 \sum_{k=0}^{p} a_k^2 \prod_{j=1}^{k} \mathbb{E}\left[ \left( w^T \widetilde{z}_{i, \frac{k(k-1)}{2}+j} \right)^2 \right] \\
&\leq \|\mathcal{Y}\|^2 R^2 \sum_{k=0}^{p} a_k^2 \prod_{j=1}^{k} \|\mathcal{Y}\|^2 r^2 \left( 1 + \frac{p(p+1)}{\mu^2} \right) \\
&= a_\infty^2 \|\mathcal{Y}\|^2 R^2 \frac{\|\mathcal{Y}\|^{2p+2} r^{2p+2} \left( 1 + \frac{p(p+1)}{\mu^2} \right)^{p+1} - 1}{\|\mathcal{Y}\|^2 r^2 \left( 1 + \frac{p(p+1)}{\mu^2} \right) - 1}
\end{aligned}
$$

and the latter can be bounded as

$$
\begin{aligned}
\mathbb{E}\left[ \|\widetilde{g} y_i \varphi^{(Nys)}(x_i) - \widetilde{g}\widetilde{z}_{i0}\|_2^2 \right] &= \mathbb{E}\left[ \widetilde{g}^2 \right] \mathbb{E}\left[ \|y_i \varphi^{(Nys)}(x_i) - \widetilde{z}_i\|_2^2 \right] \\
&\leq (\mathrm{Var}(\widetilde{g}) + (c+\alpha)^2) \mathbb{E}\left[ \|y_i \varphi^{(Nys)}(x_i) - \widetilde{z}_i\|_2^2 \right] \\
&\leq \frac{4m\|\mathcal{Y}\|^2}{\mu^2} (\mathrm{Var}(\widetilde{g}) + (c+\alpha)^2)
\end{aligned}
$$

since $|\mathbb{E}[\widetilde{g}]|=|h(w^T y_i \varphi^{(Nys)}(x_i))| \le |l_0'|+|h-l_0'| \le c+\alpha$. Finally

$$\mathbb{E}\left[\|gy_i\varphi^{(Nys)}(x_i)-\mathbb{E}_{(x,y)\sim\mathcal{P}}\left[\nabla_w l(w^T\varphi^{(Nys)}(x_i);y_i)\right]\|_2^2\right]$$

$$\le \quad 2\mathbb{E}\left[\|gy_i\varphi^{(Nys)}(x_i)-\nabla_w l(w^T\varphi^{(Nys)}(x_i);y_i)\|_2^2\right]$$

$$+2\mathbb{E}\left[\|\nabla_w l(w^T\varphi^{(Nys)}(x_i);y)-\mathbb{E}_{(x,y)\sim\mathcal{P}}\left[\nabla_w l(w^T\varphi^{(Nys)}(x_i);y)\right]\|_2^2\right]$$

$$\le \quad 2R^2\|\mathcal{Y}\|^2(\mathbb{E}\left[(g-l_0'(w^T\varphi^{(Nys)}(x_i);y))^2\right]+4c^2)$$

$$\le \quad (\alpha^2+4c^2)R^2\|\mathcal{Y}\|^2,$$

and combining the bound of $\mathrm{Var}(\widetilde{g})$ we obtain

$$\mathbb{E}\left[\|\widetilde{G}(w;\widetilde{z}_i)-\mathbb{E}_{(x,y)\sim\mathcal{P}}\left[\nabla_w l(w^T\varphi^{(Nys)}(x_i);y)\right]\|_2^2\right]\le O\left(\frac{a_\infty^2 r^{2p}p^{2p+1}}{\mu^{2p+4}}+\alpha^2+c^2\right)\frac{mR^2\|\mathcal{Y}\|^2}{\mu^2}$$

and the algorithm has $\left(4R\|\mathcal{Y}\|\alpha r, \frac{\lambda+bR^2\|\mathcal{Y}\|^2}{2}, O\left(R\|\mathcal{Y}\|\sqrt{\frac{a_\infty^2 r^{2p}p^{2p+1}}{\mu^{2p+4}}+\alpha^2+c^2}\right)\right)$ oracle. By Lemma 2, the excess empirical risk is bounded by

$$\widetilde{O}\left(\left(\alpha^2+c^2+a_\infty^2\left(\frac{rp}{\mu}\right)^{2p+1}\right)\frac{mR^2\|\mathcal{Y}\|^2\log^2\frac{1}{\beta}}{n\lambda\mu^2}+\alpha R\|\mathcal{Y}\|r\right)$$

with probability $1-\beta$. $\qquad\qquad\square$

### 6.3.10 Proof of privacy of the DP subsampling-based Nyström method

*Proof.* For a sequence $x_{i_1},\ldots,x_{i_k}$, denoted by $S$ without duplication, let $M(S)=v_S+\eta\varepsilon$ where $v_S:=[x_{i_1},\ldots,x_{i_k}]^T$ is a flattened vector obtained from the sequence with increasing index order, i.e., $i_1<\cdots<i_k$, and $\varepsilon$ be the random vector in $\mathbb{R}^{kd}$ with i.i.d. components following the Laplace distribution given in the algorithm. Note that our algorithm first release the output of $M(S_m)$ for $\eta=\frac{d}{\log\left(1+\frac{n}{m}(e^\epsilon-1)\right)}$ where $S_m$ is a random subset of size $m$ uniformly drawn from $D$, in line 2 and 3. The operations in line 4-7 does not refer to the data thus no privacy leakage. Therefore we show the algorithm $M(S_m)$ is $\epsilon$-DP if $\eta=\frac{d}{\log\left(1+\frac{n}{m}(e^\epsilon-1)\right)}$. However, we will assign the value of $\eta$ at the very last of our proof, and leave $\eta$ to be undetermined until then.

Denote $S_m, S_m'$ the length $m$ uniform random sequences of $D$ and $D'$, respectively, and $S_{m-1}$ the length $m-1$ random sequence of $D\cap D'$. Denote $\mathrm{Lap}(v,\eta)$ be the distribution of the $v+\eta\varepsilon$ for $v\in\mathbb{R}^{kd}$. If we denote the distributions of the outputs of the algorithm for $D$ and $D'$ as $\mu_D$ and $\mu_{D'}$ then

$$\mu_D \quad \sim \quad \frac{(n-m)!}{n!}\sum_{S_m\subset D}\mathrm{Lap}\left(v_{S_m},\eta\right)$$

$$\mu_{D'} \quad \sim \quad \frac{(n-m)!}{n!}\sum_{S_m'\subset D'}\mathrm{Lap}\left(v_{S_m'},\eta\right).$$

Denote

$$\mu_1 \quad = \quad \frac{(n-m-1)!}{(n-1)!}\sum_{x_n\notin S_m}\mathrm{Lap}\left(v_{S_m},\eta\right)$$

$$\mu_2 \quad = \quad \frac{(n-m)!}{m((n-1)!)}\sum_{x_n\in S_m}\mathrm{Lap}\left(v_{S_m},\eta\right)$$

$$\mu_1' \quad = \quad \frac{(n-m-1)!}{(n-1)!}\sum_{x_n'\notin S_m'}\mathrm{Lap}\left(v_{S_m'},\eta\right)$$

$$\mu_2' \quad = \quad \frac{(n-m)!}{m((n-1)!)}\sum_{x_n'\in S_m'}\mathrm{Lap}\left(v_{S_m'},\eta\right).$$

Then,

$$\mu_D = \frac{n-m}{n}\mu_1 + \frac{m}{n}\mu_2$$

$$\mu_{D'} = \frac{n-m}{n}\mu_1' + \frac{m}{n}\mu_2'.$$

Note that $\mu_1 = \mu_1'$. Thus

$$
\begin{aligned}
D_{1+\frac{m}{n}(e^\epsilon-1)}(\mu_D||\mu_{D'}) &= \int \left(\frac{n-m}{n}\mu_1 + \frac{m}{n}\mu_2 - \left(1+\frac{m}{n}(e^\epsilon-1)\right)\left(\frac{n-m}{n}\mu_1' + \frac{m}{n}\mu_2'\right)\right)_+ \\
&= \frac{m}{n}\int\left(\mu_2 - e^\epsilon\left(\left(1-\frac{1+\frac{m}{n}(e^\epsilon-1)}{e^\epsilon}\right)\mu_1' + \left(\frac{1+\frac{m}{n}(e^\epsilon-1)}{e^\epsilon}\right)\mu_2'\right)\right)_+ \\
&= \frac{m}{n}D_{e^\epsilon}\left(\mu_2||\left(1-\frac{1+\frac{m}{n}(e^\epsilon-1)}{e^\epsilon}\right)\mu_1' + \left(\frac{1+\frac{m}{n}(e^\epsilon-1)}{e^\epsilon}\right)\mu_2'\right) \\
&\leq \frac{m}{n}\max\{D_{e^\epsilon}\left(\mu_2||\mu_1'\right), D_{e^\epsilon}\left(\mu_2||\mu_2'\right)\}
\end{aligned}
$$

since $\mu_1 = \mu_1'$. Also

$$
\begin{aligned}
D_{e^\epsilon}\left(\mu_2||\mu_1'\right) &\leq \frac{(m-1)!}{(n-m)((n-1)!)}\sum_{S_{m-1}}\sum_z D_{e^\epsilon}\left(v_{S_{m-1}\cup\{x_n\}} + \eta\varepsilon||v_{S_{m-1}\cup\{z\}} + \eta\varepsilon'\right) \\
&\leq \frac{(m-1)!}{(n-m)((n-1)!)}\sum_{S_{m-1}}\sum_z \delta_\epsilon\left(\frac{z-x_n}{\eta}\right)
\end{aligned}
$$

where $z$ are selected uniformly in $D' - S_{m-1} - \{x_n'\}$, $\varepsilon$ and $\varepsilon'$ are independent random vectors with i.i.d. components following Laplace distribution given in the algorithm, and $\delta_\epsilon(c)$ denotes $D_{e^\epsilon}(c + \varepsilon||\varepsilon')$. Although we refer to $S_{m-1}, \{x_n\}, \{z\}$ as sets for convenience, they are actually sequences. $\{x_n\}$ or $\{z\}$ can be added at any position in $S_{m-1}$, and each such case should be treated differently. However, by comparing the case where $x_n$ is added to the $i$th position of $S_{m-1}$ with the case where $z$ is added to the $i$th position of $S_{m-1}$, we can reach the desired conclusion.

It is known that if the $\ell_1$ norm difference of the output of an deterministic algorithm $\mathcal{A} : \mathcal{D} \to \mathbb{R}^d$ is at most $\epsilon$ then $\mathcal{A}(D) + \varepsilon$ is $\epsilon$-DP (Dwork et al., 2006b) or equivalently $D_{e^\epsilon}(c + \varepsilon||\varepsilon') = 0$ if $||c||_1 = \epsilon$ for vector $c$. Thus if we set $\eta = \frac{d}{\epsilon}$ then the inequality gives $D_{e^\epsilon}(\mu_2||\mu_1') = 0$.

Thus, the algorithm is $\log\left(1 + \frac{m}{n}(e^\epsilon - 1)\right)$-DP if $\eta = \frac{d}{\epsilon}$. Replacing $\epsilon$ to $\log\left(1 + \frac{n}{m}(e^{\frac{\epsilon}{2}} - 1)\right)$ we arrive at the conclusion that the algorithm is $\frac{\epsilon}{2}$-DP if $\eta = \frac{d}{\log\left(1+\frac{n}{m}(e^{\frac{\epsilon}{2}}-1)\right)}$ so we are done. $\qquad\square$

